# Randomized Sparse Matrix Compression for Large-Scale Constrained Optimization in Cancer Radiotherapy

**Shima Adeli**[1]     **Mojtaba Tefagh**[1,2] *     **Gourav Jhanwar**[3]     **Masoud Zarepisheh**[3]

[1]Sharif University of Technology
[2]University of Edinburgh
[3]Memorial Sloan Kettering Cancer Center

## Abstract

Radiation therapy, treating over half of all cancer patients, involves using specialized machines to direct high-energy beams at tumors, aiming to damage cancer cells while minimizing harm to nearby healthy tissues. Customizing the shape and intensity of radiation beams for each patient leads to solving large-scale constrained optimization problems that need to be solved within tight clinical time-frame. At the core of these challenges is a large matrix that is commonly sparsified for computational efficiency by neglecting small elements. Such a crude approximation can degrade the quality of treatment, potentially causing unnecessary radiation exposure to healthy tissues—this may lead to significant radiation-induced side effects—or delivering inadequate radiation to the tumor, which is crucial for effective tumor treatment. In this work, we demonstrate, for the first time, that randomized sketch tools can effectively sparsify this matrix without sacrificing treatment quality. We also develop a novel randomized sketch method with desirable theoretical guarantees that outperforms existing techniques in practical application. Beyond developing a novel randomized sketch method, this work emphasizes the potential of harnessing scientific computing tools, crucial in today's big data analysis, to tackle computationally intensive challenges in healthcare. The application of these tools could have a profound impact on the lives of numerous cancer patients. Code and sample data available at https://github.com/PortPy-Project/CompressRTP

## 1 Introduction

In 2020, an estimated 18.1 million new cancer cases and about 9.9 million cancer-related deaths were reported globally [22]. Radiation therapy (RT) is integral to cancer treatment, utilized in approximately half of all cases, either alone or in combination with other treatments like surgery or chemotherapy [22]. RT involves using specialized machines to direct high-energy radiation beams at tumors, with the primary goal of destroying cancer cells while minimizing damage to healthy tissues. This process requires precise optimization of machine parameters, such as beam shapes and angles, tailored to each patient's unique anatomy. It involves solving large-scale, constrained, non-linear optimization problems swiftly within clinical time constraints [22, 19]. The urgency of this task is heightened in modern online adaptive radiotherapy techniques, where rapid solution is essential since patients remain immobilized on the treatment couch during preparation [12]. Delays not only compromise patient comfort but can also affect treatment outcomes, as any rapid anatomical changes (e.g., bladder filling in prostate cancer) can render treatment plans based on initial anatomy sub-optimal. Thus, quickly solving these optimization problems is crucial.

We briefly describe the mathematical modeling of radiotherapy treatment (see [22, 19] for more details). The patient's body is discretized into small three-dimensional voxels (indexed by $i = 1, \ldots, m$), and each radiation beam is discretized into small two-dimensional beamlets (indexed by $j = 1, \ldots, n$). The radiation dose delivered to each voxel $i$ from each beamlet $j$ with unit intensity is precalculated and represented by $a_{ij}$, forming a matrix $A$ — commonly referred to as the *dose influence matrix*. This matrix is typically large, containing about 100,000 to 500,000 rows corresponding to the patient's voxels, and 1,000 to 20,000 columns representing the beamlets of the radiotherapy machine [18]. Our objective is to optimize the intensities of the beamlets, denoted by $x$, in order to achieve a desired radiation dose, $Ax$, that is delivered to the patient's body. For the tumor voxels, we aim to achieve a radiation dose that approximates the dose prescribed by a physician while for healthy voxels we aim to minimize the radiation dose as much as possible. The optimization problem can be described in the following general form:

$$
\begin{aligned}
\text{minimize} \quad & f_0(Ax) + f_1(x) \\
\text{subject to} \quad & g(Ax) \leq 0 \\
& h(x) \leq 0,
\end{aligned}
\tag{1}
$$

where, $f_0(Ax)$ measures the quality and 'goodness' of the radiation dose $Ax$, $f_1(x)$ assesses the quality of the beamlet intensities. The functions $g$ and $h$ represent constraints on the dose received by the voxels and the intensities of the beamlets, respectively. Various formulations for these functions have been suggested in existing research [17]; however, the following quadratic optimization problem has arguably been the most commonly used formulation [22, 19]:

$$
\begin{aligned}
\text{minimize} \quad & \sum_{s \in \bar{S}} \sum_{i \in I_s} (w_+^s \max(A_i^s x - d^s, 0)^2 + w_-^s \max(d^s - A_i^s x, 0)^2) + ||Px||_2^2 \\
\text{subject to} \quad & A_i^s x \leq d_{Max}^s, \forall s \in \bar{S}, i \in I_s \\
& Mean(A^s x) \leq d_{Mean}^s, \forall s \in \bar{S} \\
& x \geq 0,
\end{aligned}
\tag{2}
$$

where, $\bar{S}$ represents the set of structures (i.e., organs, tumors), including tumor and healthy structures, and $I_s$ represents the set of voxels belonging to structure $s$. The first term in the objective function is a two-sided quadratic function penalizing the radiation overdose/underdose with the penalty weight $w_+^s/w_-^s$, where the radiation overdose/underdose is defined as a delivered radiation dose, $A^s x$, exceeding/below the prescribed dose, $d^s$, for each structure $s \in \bar{S}$. For tumor-affiliated structures, the prescribed dose is given by a radiation oncologist. For healthy structures, the prescribed dose is zero, $d^s = 0$, and there is no underdose penalty, $w_-^s = 0$. The second term in the objective function, $f_1(x) = ||Px||_2^2$, aims to penalize variations in intensities across neighboring beamlets to promote smoothness in beamlet intensities for enhanced radiation delivery (each row of matrix $P$ has a value of 1 and -1 for two neighboring beamlets). The first/second set of constraints impose maximum/mean dose constraints to satisfy the maximum/mean dose limits defined by a clinical protocol for each structure $s \in \bar{S}$, and the last constraint is a physical non-negativity constraint on the beamlet intensities. The overdose/underdose penalty weights $w_+^s/w_-^s$ need to be adjusted for each patient and various techniques have been developed to automate this process (see [22] and references therein).

We will use the second formulation (Eq. 2) for our experiments in this study. However, regardless of the specific functions chosen in Problem 1 and the technique used to adjust the problem hyper-parameters, the size and structure of matrix $A$ are crucial in determining the computational intensity of the problem. Considering that these large-scale, non-linear, constrained optimization problems are often solved using the interior-point method with cubic computational complexity, the computational time can increase significantly with the size and density of non-zero elements in matrix $A$. Therefore, matrix sketching could be a compelling choice to improve the computational complexity of these problems. There has been a body of research employing matrix sketching, often using a transformation matrix $S$, resulting in $SA$ with a reduced number of rows, to improve the computational efficiency of the least-squares optimization problems commonly arising in machine learning applications [14, 16, 21]. However, these sketching techniques cannot be used where matrix $A$ is also involved in the constraints, as is the case in radiotherapy applications. Reducing the number of rows would prevent direct access to $Ax$, which is crucial for evaluating $g(Ax) \leq 0$ in Problem 1. Thus, we explore the potential of using matrix sparsification, a specific form of matrix sketching, to substitute the dose influence matrix $A$ with a sparse matrix $S$ to improve the computational efficiency of our

radiotherapy constrained optimization problems. In fact, a very simple form of matrix sparsification is currently being used in practice, where all small elements of the matrix below a predefined threshold, typically less than $0.01 \times \max(A)$, are discarded [10]. This method, which we will refer to as the "naive" approach, is clearly not the most optimal solution and may adversely affect the quality of the treatment. The primary concern with this approach is that the radiation dose calculated and optimized in Problem 1 using the modified matrix $A$ may not accurately reflect the actual dose received by the patient. This discrepancy arises from the inherent inaccuracies introduced by the truncation of matrix $A$, which could potentially lead to sub-optimal treatment outcomes.

In this study, we demonstrate that using matrix sparsification techniques, primarily developed by the machine learning community, we can enjoy the computational efficiency of working with sparse matrices and still being able to solve constrained optimization problems within the clinical timeframe, without significantly compromising the integrity of the original problem that could potentially degrade the treatment quality. Matrix sparsification techniques carefully sample and scale the elements of the original dense matrix $A$ to create a sparse sketch matrix $S$ that minimizes $||A - S||_2$. Prior research predominantly utilized matrix sparsification for applications such as low-rank approximation and principal component analysis (PCA) [1, 2]. This study uniquely demonstrates the utility of matrix sparsification for efficiently addressing large-scale, constrained, nonlinear optimization challenges within constrained timeframes. We demonstrate that applying a randomized sketch to the influence matrix $A$ in radiotherapy optimization significantly outperforms the current naive sparsification approach. To the best of our knowledge, this is the first application of matrix sparsification with a publicly available benchmark dataset, encouraging further research in this direction. Furthermore, we have developed a novel randomized sketching algorithm that exhibits superior performance compared to existing techniques and is supported by theoretical guarantees for its efficacy. As formally stated in Theorem 3.6, Lemma 3.7, and Theorem 3.9, the proposed algorithm ensures a minimal impact on the constraints, objective function and the optimal points of the original optimization Problem 2.

## 2 Related Work

In matrix sparsification, we typically aim for an unbiased approximation of a matrix $A$ by another matrix $S$ such that $||A - S||_2 \leq \tau$ for a given $\tau > 0$, while minimizing the number of nonzero entries in $S$ [1, 2, 3, 7, 8, 15]. The $\ell_2$ norm of the difference between $A$ and $S$ serves as a measure of this error, a choice that has been justified in the literature [2] for various applications. The sparsification techniques proposed in the literature fall into two categories. The first involves randomly scaling each matrix entry independently. Specifically, for an entry $a_{ij}$ it is scaled to $a_{ij}/p_{ij}$ with probability $p_{ij}$, or is set to zero otherwise [1, 3, 8, 15]. This process increases the magnitude of certain matrix entries and zeros out others, resulting in a sparse matrix. Importantly, the resulting matrix serves as an unbiased estimator of the original matrix, with its entries acting as independent random variables. The probability typically follows the formula $p_{ij} = cf(a_{ij})$, where $c > 0$ is a universal constant. This method, first introduced by Achlioptas and McSherry [1], involves scaling each entry $a_{ij}$ to $a_{ij}/p_{ij}$ with probability $p_{ij} = ca_{ij}^2$; otherwise, the entry is set to zero. This technique, referred to as $\ell_2$ scaling, laid a foundational basis in the domain of matrix sparsification. However, Achlioptas and McSherry [1]'s approach faced limitations in theoretical guarantees and required a significant number of non-zero entries to achieve a satisfactory $\ell_2$ norm bound. Additionally, scaling small entries could disproportionately inflate values in the resulting matrix. Building upon this, Arora et al. [3] introduced a variation focused on deterministically retaining the largest entries in the matrix while randomly scaling the smaller ones. During the scaling phase, each matrix entry is scaled to $a_{ij}/p$ with a probability $p = c|a_{ij}|$, or set to zero otherwise. This method is fast in practice as it requires only a single pass over all the non-zero entries and Arora et al. [3] demonstrated that their method outperforms Achlioptas and McSherry [1] approach. However, this approach substitutes all small entries, which are not reduced to zero, with a constant ($c$ or $-c$). This could significantly affect its performance, especially when the matrix undergoes extensive sparsification.

The second category of sparsification techniques, introduced later, involves independently sampling from the entries of $A$ using a probability distribution $p$. Each sample generates a matrix filled with zeroes, except for the sampled entry. Subsequently, $s$ samples are collected, and their average forms a sparse approximation. Contrary to the first approach, the entries of the resulting matrix $S$ are not independent. Instead, $S$ is formed by summing independent random matrices. To ensure that the resulting matrix is an unbiased estimator of $A$, the sampled matrix's entry should be $a_{ij}/p_{ij}$, where

$p_{ij}$ is derived from the probability distribution $p$. In this context, the choice of probability distribution $p$ is critical. Drineas and Zouzias [7] introduced a technique where $p_{ij}$ is proportional to $a_{ij}^2$, termed $\ell_2$ sampling, and used Bernstein inequality [20] to calculate the number of samples needed to achieve a desired accuracy. Later, Achlioptas et al. [2] introduced a near-optimal probability distribution under specific conditions. Additionally, Braverman et al. [4] developed a near-optimal method specifically designed for numerically sparse matrices (this method belongs to the first category). While the theoretical bounds of this method closely align with those of Achlioptas et al. [2], the approach by Braverman et al. [4] is applicable to all matrices.

The primary issue with the second category of techniques is the increase in the required number of non-zero elements in the sparse matrix, which grows in proportion to the inverse square of the error, $1/\tau^2$, unlike the $1/\tau$ growth rate seen in the first category. This is especially problematic given that the interior point methods typically used to solve Problem 1 have cubic computational complexity, which results in a cubic increase in processing time with the size of the input data. However, the second category may be preferable in applications when data is only accessible in a streaming manner. The first category necessitates access to the complete dataset. In this study, we assume that the matrix $A$ is precomputed—a time-intensive process that often takes as long as solving the optimization problems themselves. Theoretically, these matrices could be computed on a row-by-row basis [6], allowing the sparsification algorithm to operate concurrently, provided it can handle streaming data. Our proposed hybrid approach can, in principle, be used in a streaming manner [2].

**Notations.** Let $a_{ij}$ denote the element in the $i$-the row and $j$-th column of matrix $A$, and let $\mathrm{nnz}(A)$ denote the number of non-zero entries in $A$. For matrix $A$, we consider the entry-wise $\ell_1$ norm, defined as $\|A\|_1 = \sum_{i=1}^{m} \sum_{j=1}^{n} |a_{ij}|$, and the spectral norm defined as $\|A\|_2 = \max_{\|x\|_2=1} \|Ax\|_2$. Additionally, we introduce $A_{(i)} \in \mathbb{R}^{m \times n}$, a matrix with all zero entries except for the $i$-th row, which remains identical to the $i$-th row of $A$. Finally, for a vector $a$, we define its numerical sparsity as $\mathrm{ns}(a) = \min\{k \geq 0 : \|a\|_1 \leq \sqrt{k} \|a\|_2\}$ and let $\mathrm{ns}(A)$ denote the maximum numerical sparsity of its rows and columns.

## 3 Algorithm Description

In this study, we introduce an algorithm that matches the theoretical requirements of the first category for the number of non-zero elements (i.e., the growth of the non-zero elements proportional to $1/\tau$), while outperforming existing techniques in practical experiments. Our findings reveal that this approach markedly enhances the accuracy of sparse sketches for large matrices that appear in the context of cancer radiotherapy optimization. This method can be seen as a hybrid, combining advantages from both categories. We deterministically retain the largest entries, akin to the strategy suggested by Arora et al. [3]. This method is particularly relevant for our application, as the matrices appearing in radiotherapy exhibit a distribution closely resembling an exponential curve. Consequently, the number of elements exceeding any given threshold remains significantly small relative to the total matrix size. This phenomenon occurs because radiation delivered from each individual beamlet (corresponding to a matrix column) directly deposits radiation to a limited number of voxels (corresponding to matrix rows), resulting in a few large matrix entries. However, radiation also scatters, delivering smaller doses to additional voxels throughout the body, leading to small values across all voxels. In contrast to the approach by Arora et al. [3], which involves substituting all minor non-zeroed-out entries with $\pm c$, our strategy seeks to counterbalance the effects of sparsification on a row-by-row basis by redistributing the sum of all minor elements within that row. This method is designed to preserve the integrity of each row, taking into consideration the diverse distribution patterns across the matrix's rows. Specifically, in our application, rows corresponding to tumor voxels often exhibit higher values due to their position in the direct path ("cross-fire") of radiation, receiving more substantial doses. Applying a uniform value for all substituted entities, in this context, results in a more pronounced approximation for tumor voxels, potentially compromising the accuracy of the final dose delivered to the tumor, as our results will demonstrate. After isolating the larger elements, we apply $\ell_1$ sampling to the remaining entries in each row independently. This process generates a random matrix, essentially the aggregate of several random matrices, which more closely aligns with the practices of the second category. However, unlike conventional sampling methods (e.g., [7]), our technique automatically identifies the optimal number of samples for each row, consistently less than the total number of columns, eliminating the necessity for tuning sampling parameters.

Our method is nearly as fast as that proposed by Arora et al. [3] but provides superior accuracy, particularly beneficial when high levels of sparsification needed. This is crucial in our application, where approximately 96-98% sparsification is required to solve the large-scale non-linear constraint optimization problems within a feasible clinical timeframe. Our algorithm, described in *Algorithm 1*, is named the Randomized Minor-value Rectification (RMR), as it preserves the larger values while rectifying a random selection of smaller values to offset those that have been zeroed out.

---

**Algorithm 1** Randomized Minor-value Rectification (RMR)

---

**input** $A \in \mathbb{R}^{m \times n}$: The matrix to be sparsified, $\epsilon$: The threshold for sparsification
**output** $S$: The sparsified matrix
  $S = $ deep copy of $A$
  **for** each row $i$ in $\{1, 2, \ldots, m\}$ **do**
    $T_i = \{j \mid 0 < |a_{ij}| \le \epsilon\}$
    set $s_{ij} = 0$ for all $j \in T_i$
    $\Sigma_i = \sum_{j \in T_i} |a_{ij}|$
    $k_i = \lceil \Sigma_i / \epsilon \rceil$
    **for** $t = 1, 2, \ldots, k_i$ **do**
      randomly select $j \in T_i$ (with probability proportional to $|a_{ij}|$) and update $s_{ij} \leftarrow s_{ij} + \Sigma_i / k_i \times \text{sign}(a_{ij})$
    **end for**
  **end for**

---

**Lemma 3.1.** *For any matrix $A \in \mathbb{R}^{m \times n}$ the number of samples taken by the RMR algorithm is bounded above by* $\text{nnz}(A)$.

The $\text{nnz}(A)$ upper bound in the above lemma, which also applies to the algorithm proposed by Arora et al. [3], may not seem interesting at first; however, the number of samples could be much larger than $\text{nnz}(A)$ for other algorithms, as each entry of the matrix could be sampled multiple times. The following theorems provide the theoretical guarantees of the RMR algorithm (proof in Appendix A).

In the remainder of this section, we assume that the sparse matrix $S$ is obtained by applying Algorithm 1 to $A$. Instead of solving the original optimization problem 2 directly, we substitute the dense matrix $A$ with the sparse matrix $S$, thereby formulating an approximated optimization problem that we refer to as the *surrogate problem*. Let $x_A$ and $x_S$ denote arbitrary optimal points for the original and surrogate optimization problems, respectively.

**Theorem 3.2** (Absolute $\ell_2$-norm error). *Given a matrix $A \in \mathbb{R}^{m \times n}$, it can be shown that:*

(a) *The number of nonzero entries of $S$ is less than $m + \|A\|_1 / \epsilon$.*

(b) *For any given $\delta, \tau > 0$, by setting*

$$\epsilon = \frac{\tau \left( \sqrt{1 + \frac{9 \max(m, n-1)}{\log \frac{m+n}{\delta}}} - 1 \right)}{3\sqrt{2} \max(m, n-1)},$$

*it follows that $\mathbb{P}\left\{ \|A - S\|_2 \ge \tau \right\} \le \delta$.*

**Corollary 3.3** (Absolute $\ell_2$-norm error). *In Algorithm 1, by setting*

$$\epsilon = \frac{\tau}{4\sqrt{\max(m, n-1) \log(m+n)}}$$

*the resultant matrix $S$ will contain no more than $m + 4\sqrt{\max(m, n-1) \log(m+n)} \, \|A\|_1 / \tau$ nonzero entries, and we have $\mathbb{P}\left\{ \|A - S\|_2 \ge \tau \right\} \le 1/(m+n)$.*

Theorems 3.6 and 3.9 establish theoretical guarantees and provide bounds on the discrepancies between the original and surrogate optimization problems with respect to their constraints and objective functions, respectively. In essence, applying the RMR algorithm to sparsify the matrix and subsequently solving the surrogate problem yields a near-optimal solution to the original problem.

**Lemma 3.4** (Row-wise error). *For every vector $x \in \mathbb{R}^n$, the probability that the absolute value of the $i$-th entry of the vector $(A - S)x$ exceeds $c\epsilon \|x\|_2$ is less than $2\exp(-c^2/(4 + 2\sqrt{2}c/3))$, i.e.,* $\mathbb{P}\left\{ |((A - S)x)_i| \ge c\epsilon \|x\|_2 \right\} \le 2\exp(-c^2/(4 + 2\sqrt{2}c/3))$.

Note that the upper bound obtained from Lemma (3.4) is significantly tighter than the bound derived directly from Corollary (3.3). To understand this, consider the following inequality:

$$|((A - S)x)_i| \leq \|(A - S)x\|_2 \leq \|A - S\|_2\|x\|_2 \leq 4\sqrt{\max(m, n-1)\log(m+n)}\epsilon\|x\|_2$$

with probability at least $1 - 1/(m+n)$, as stated in (3.3). However, if we set $c = 5\log(m+n)$, we achieve the same failure probability, but the upper bound on the error becomes $5\log(m+n)\epsilon\|x\|_2$, which is of a better order.

**Lemma 3.5** (Feasibility gap). *Let $C \in \mathbb{R}^{k \times m}$ be a normalized matrix where the $l_1$-norm of each row is less than or equal to one, and assume that for an arbitrary $x \in \mathbb{R}^n$, there exists $u \in \mathbb{R}^k$ for which we have $CSx \leq u$. Then, with a probability of 0.95, the maximum violation of any constraint in $CAx \leq u$ is bounded above by $(19 + 5\log m)\epsilon\|x\|_2$. Conversely, if $CAx \leq u$, then the maximum violation of any constraint in $CSx \leq u$ is bounded above by $(19 + 5\log m)\epsilon\|x\|_2$ with a probability of 0.95.*

The feasible region of the optimization problem (2) satisfies the assumptions of Lemma 3.5, leading to the following theorem.

**Theorem 3.6** (Feasibility gap for Problem 2). *An optimal point of the original problem, $x_A$, violates each constraint of the surrogate problem by no more than $(19 + 5\log m)\epsilon\|x\|_2$ with a probability of at least 0.95. Conversely, an optimal point of the surrogate problem, $x_S$, violates each constraint of the original problem by no more than $(19 + 5\log m)\epsilon\|x\|_2$ with the same probability.*

**Lemma 3.7** (Objective function discrepancy). *The absolute discrepancy between the objective functions of the original and surrogate problems does not exceed $e\left(f_0(Ax) + m(1+e)\sum_{s \in \bar{S}}\left(w_+^s + w_-^s\right)\right)$ with a probability of at least 0.95, where $e = (19 + 5\log m)\epsilon\|x\|_2$.*

**Lemma 3.8.** *Suppose that for an $x \in \mathbb{R}^n$ satisfying the convex constraints $h(x) \leq 0$, the maximum violation of any convex constraint $g_i \leq 0$ in the optimization problem (1) is bounded above by $e \geq 0$, i.e., $g_i(Ax) \leq e$. Additionally, assume there exists an interior point of the feasible set $\tilde{x} \in \mathbb{R}^n$ that is strictly feasible by a margin of at least $s > 0$ for each constraint $g_i \leq 0$, i.e., $g_i(A\tilde{x}) \leq -s$. Then, there exists a feasible point $\hat{x} \in \mathbb{R}^n$ such that:*

$$\|x - \hat{x}\| \leq \frac{e}{s}(\|x\| + \|\tilde{x}\|)$$

**Theorem 3.9** (Sub-optimality gap for Problem 2). *An optimal point of the surrogate problem, $x_S$, is a near-optimal solution to the original problem with a probability of at least 0.95, and the sub-optimality gap of $O(e)$, where*

$$e = (19 + 5\log m)\epsilon \max\left(\|x_A\|_2, \|x_S\|_2\right).$$

*In other words, we have:*

$$[f_0(Ax_S) + f_1(x_S)] - [f_0(Ax_A) + f_1(x_A)] = O(e).$$

*Remark* 3.10. Note that the proof of Theorem (3.9) can be generalized to any matrix approximation scheme with a bounded error norm for the general convex optimization problem (1), given the key assumptions that the objective function is Lipschitz continuous and that there exists a strictly feasible point with a lower bounded slackness for all approximated constraints. These assumptions are crucial. If the objective function is not Lipschitz continuous, the error in matrix approximation can significantly alter the value of the objective function, thereby drastically affecting the optimal point. Additionally, if the assumption of Lemma (3.8) is not satisfied and no interior point exists for the approximated constraints, the feasible set could be reduced to a single point in extreme cases. Even with an arbitrarily small error in approximating the constraints, the feasible set might vanish, rendering the approximated problem infeasible.

Now we conduct a comparative analysis of the performance of the RMR algorithm against the algorithms proposed by Arora et al. [3], Drineas and Zouzias [7], Achlioptas et al. [2], and Braverman et al. [4] denoted as "AHK06", "DZ11", "AKL13", and "BKKS21" respectively. Table 1 presents the sparsity of the resulting matrix $S$ for each considered method, adhering to the constraint $\|A - S\|_2 \leq \tau$. The computational time required to solve the constrained optimization Problem 2 is strongly correlated with the sparsity of the matrix, as also verified by our computational experiments (see Figure 1). Therefore, we use sparsity as a proxy for computational efficiency. Although the values

Table 1: Comparison of Theoretical Guarantees Across Various Algorithms.

| Method | Number of non-zero elements | Failure Probability |
|--------|------------------------------|---------------------|
| AHK06[3] | $O\left(\sqrt{m+n}\,\|A\|_1/\tau\right)$ | $\exp(-\Omega(m+n))$ |
| DZ11[7] | $28(m+n)\log\left(\sqrt{2}(m+n)\right)\|A\|_F^2/\tau^2$ | $1/(m+n)$ |
| AKL13[2] | $O\left(\log(n)\sum_{i=1}^{m}\|A_{(i)}\|_1^2/\tau^2 + \sqrt{\log(n)}\,\|A\|_1/\tau\right)$ | $1/10$ |
| BKKS21[4] | $O\left(\log(m)\mathrm{ns}(A)\|A\|_F^2/\tau^2 + \log(m)\,\|A\|_F\,\sqrt{\mathrm{ns}(A)n}/\tau\right)$ | Not specified |
| RMR | $m + 4\sqrt{\max(m,n-1)\log(m+n)}\,\|A\|_1/\tau$ | $1/(m+n)$ |

in Table 1 are not directly comparable, for DZ11, AKL13, and BKKS21 the number of non-zero elements scales inversely with the square of the error rate (i.e., $\frac{1}{\tau^2}$), while for RMR and AHK06, it is determined by the inverse of the error rate (i.e., $\frac{1}{\tau}$). As also confirmed by our computational experiments (Figure 1), this makes RMR and AHK06 algorithms particularly beneficial at lower error rates, which is needed for our application. Furthermore, due to the dependence of the BKKS21 on the numerical sparsity of the matrix, and considering that the matrices we tested exhibit low numerical sparsity, this algorithm also performs well across various metrics. In terms of the runtime and computational complexity of the algorithms themselves, it mainly depends on the number of required samplings. For DZ11 and AKL13, the number of required samplings also depends on the error rate, while for AHK06, BKKS21, and RMR, the dependency is solely on $\mathrm{nnz}(A)$. This also gives AHK06, BKKS21, and RMR an edge in runtime, particularly for small error rates, as confirmed by our experiments in Figure 1.

## 4 Experiments

Our objective is to illustrate that employing a randomized sparse sketch of the large *dose influence matrix* in radiotherapy markedly surpasses the existing "naive" method used in practice, which simply sparsifies the matrix by neglecting all minor elements. Additionally, we demonstrate the superior performance of our RMR algorithm over the randomized sketch techniques introduced by Arora et al. [3] ("AHK06"), Drineas and Zouzias [7] ("DZ11"), Achlioptas et al. [2] ("AKL13") and Braverman et al. [4] ("BKKS21").

**Dataset.** Our analysis utilized real-world data recently made publicly available through the open-source package PortPy [10]. We conducted experiments on data from 10 randomly selected lung patients, with detailed information provided in the Appendix (Table 2). The dose influence matrices in PortPy were derived from an FDA-approved commercial treatment planning system, Varian Eclipse$^{\text{TM}}$, using its Application Programming Interface (API). Further detailed information about the data can be found on PortPy's GitHub page. Furthermore, we conducted experiments using data from five prostate patients, which are not publicly available. The results of these experiments are included in the Appendix.

**Experiment Settings.** Due to the inherent randomness of all the algorithms, except the naive one, each experiment was repeated 5 times to assess and report variations and the algorithms' robustness. Each algorithm has a hyper-parameter threshold that determines the sparsity of the output matrix. To ensure a fair comparison, we ran each algorithm with various threshold values and compared the results based on different levels of sparsity in the output matrix (e.g., runtime of algorithm 1 vs. algorithm 2 for sparsity levels x, y). The experiments were conducted on a dual CPU system (Intel(R) Xeon(R) 6248 2.5GHz) running Windows 10 with 128 GB of RAM. For each patient, the optimization problem 2 was solved using the penalty weights, $w_+^s/w_-^s$, recommended by the PortPy package [10], along with the maximum and mean dose constraints presented in Table 3 in Appendix. All optimization problems were modeled and solved using CVXPY [5] and MOSEK [13]. To report the optimality gap, we needed to solve the original optimization problems using the original matrix $A$. However, we encountered limited memory errors on the PC. Consequently, we solved the original optimization problems on a powerful high-performance computing (HPC) system with approximately

320 GB of memory, which was used exclusively to obtain the optimal solution $x_A$ for optimality gap comparisons.

**Code availability.** Our code, which includes implementations of all models and experiment configurations, is available at https://github.com/PortPy-Project/CompressRTP.

## 4.1 Evaluation Metrics

The algorithms are compared using the following metrics:

**Relative $\ell_2$-Norm Error.** Defined as $\|A - S\|_2 \,/\, \|A\|_2$, this metric, widely adopted in existing literature, indicates how closely the sparse matrix approximates the original matrix for a given level of sparsity with respect to the spectral norm.

**Relative Optimally Gap.** This metric quantifies the sub-optimality resulting from solving the approximated problem by measuring the relative difference between the optimal objective value of the original problem (solved on HPC) and the objective value of the original problem using the optimal solution of the approximated problem. It is defined as ($\frac{(f_0(Ax_A)+f_1(x_A))-(f_0(Ax_S)+f_1(x_S))}{f_0(Ax_A)+f_1(x_A)}$).

**Feasibility Gap.** This metric measures how much the constraints of the original optimization problem (i.e., $g(Ax) \leq 0$) are violated when solving the approximated problem. It is defined as $\|\max(g(Ax_S), 0)\|_2$.

**Relative Dose Discrepancy.** This metric, which is particularly relevant in the context of radiotherapy applications, is defined as $\frac{\|Ax_S - Sx_S\|_2}{\|Ax_S\|_2}$ and quantifies the discrepancy between the radiation dose computed using the sparse optimization problem, $Sx_S$, and the actual dose received by the patient, $Ax_S$.

**Runtime.** This includes the runtime of the sparsification algorithms and the runtime of the constrained optimization problems. It is worth noting that, in practice, the constrained optimization problem needs to be solved multiple times for each patient, depending on the technique used to adjust the hyper-parameter penalty weight.

**Discrepancies in Dose Volume Histograms (DVH).** DVH plots are two-dimensional graphs specific to radiotherapy applications and are reported in the Appendix (Figures 7 and 8). These plots, extensively used by clinicians, show how much radiation is delivered to different volumes of organs (e.g., 10 Gray is delivered to at least 30% of the esophagus). Discrepancies between these plots for $Ax_S$ and $Sx_S$ are illustrated for different algorithms.

## 4.2 Results

Figure 1 provides detailed comparisons with respect to the different levels of sparsity in the output sparse matrix for one patient. Additional comparisons for the remaining nine patients can be found in the Appendix (Figures 4 to 6). The Naive algorithm is excluded in some of the plots due to its poor performance and significant deviation from other methods (impeding a meaningful comparison of others). The incomplete range coverage in the plots for DZ11 is due to this algorithm exceeding our predefined maximum time limit of 10 minutes for some of its threshold values. The standard deviation band is plotted for all metrics (over 5 runs), however, due to the robust performance of the algorithms, the bands are not visible except in the feasibility gap plot.

As demonstrated in Figure 1 (top-left plot), all randomized algorithms surpass the current naive approach in balancing accuracy and sparsity. Among these, AHK06, BKKS21, and RMR stand out, exhibiting superior performance. While AHK06 and BKKS21 marginally surpasses RMR in terms of the $\ell_2$ norm, RMR significantly exceeds AHK06 and BKKS21 in terms of feasibility and optimality gaps as well as the dose discrepancy error. This is further supported by reduced discrepancies among DVH plots reported in the Appendix (Figures 7 and 8).

In terms of computational efficiency, the naive approach unsurprisingly leads in speed due to its simple implementation, with AHK06 following closely behind. However, the relatively longer runtime

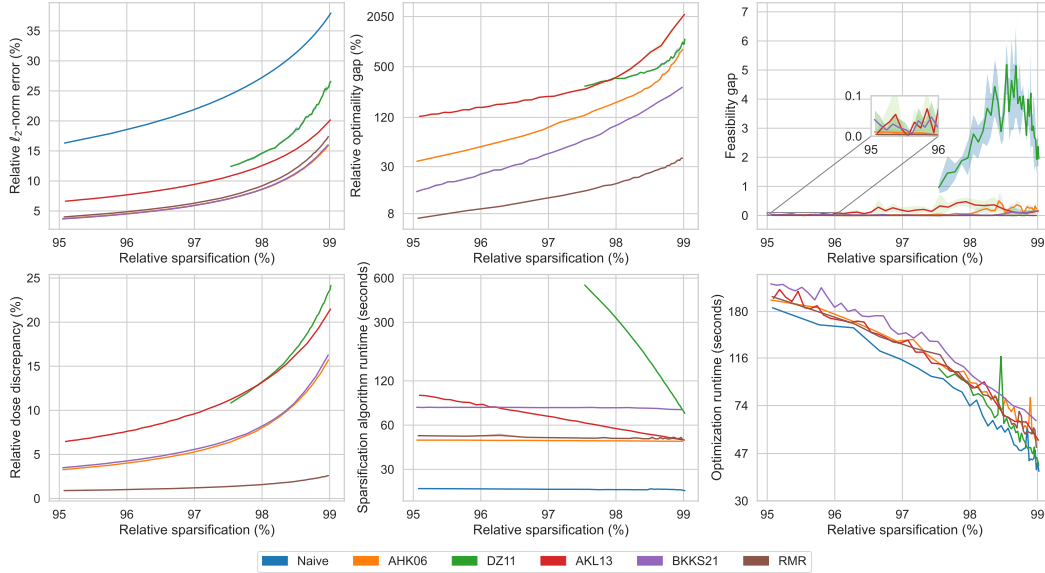

Figure 1: *(One patient, various sparsification levels)* The performance of different algorithms in terms of: relative $\ell_2$-norm error, relative optimality gap, feasibility gap, relative dose discrepancy, algorithm runtime, and optimization runtime, for a lung patient.

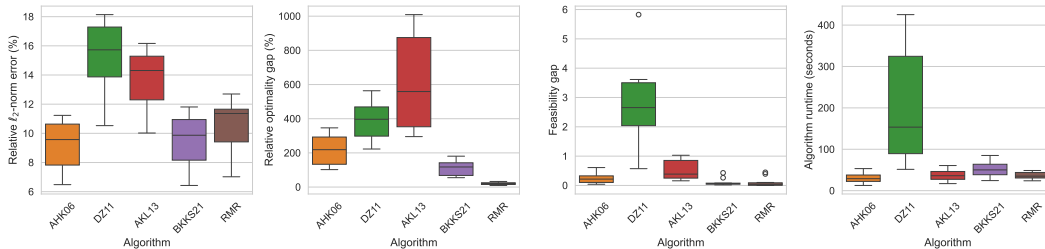

Figure 2: *(Ten patients, one sparsification level)* The performance of different algorithms in terms of: relative $\ell_2$-norm error, relative optimality gap, feasibility gap, and algorithm runtime, for ten lung patients.

of the RMR algorithm should not be seen as a significant limitation. The primary computational challenge lies in solving the optimization problem 2, which often needs to be solved multiple times for hyper-parameter tuning. For example, to achieve a relative optimality gap of 35% for patient 1, the RMR algorithm necessitates a sparse matrix with 98.90% relative sparsification (approximately 2,850,000 non-zero elements), in contrast to the 95.05% sparsification (about 12,750,000 non-zero elements) required by AHK06. Consequently, solving the optimization problem 2 with RMR's sparse sketch is considerably faster, taking 57 seconds, compared to 200 seconds for the sparse sketch produced by AHK06. The bottom-right plot illustrates the strong correlation between the number of non-zero elements in the matrix and the computational time of the constrained optimization problem, indicating that relative sparsification serves as an excellent proxy for the computational time of the optimization problems.

Figure 2 offers a high-level comparison across all patients at a fixed relative sparsity level of 98%, which can also be interpreted as a fixed computational time for the constrained optimization problem. Confirming the results of Figure 1 for more patients, this figure demonstrates the superior performance of AHK06 in terms of the $\ell_2$ norm error and algorithm runtime, while highlighting the significant advantage of the RMR algorithm in reducing optimality and feasibility gaps.

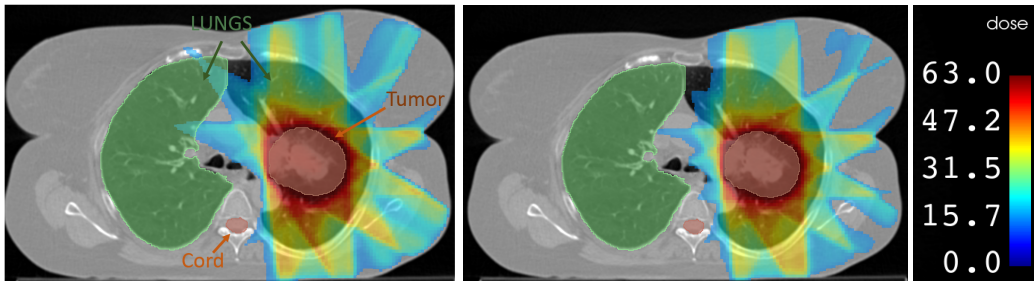

Figure 3: Radiation dose maps: Naive (left), RMR (middle) and dose labels (right)

Figure 3 presents a qualitative comparison between treatment plans generated using the naive approach (left) and RMR (middle), based on the radiation dose map commonly used by clinicians. The dose map visually represents the distribution of radiation, with color-coding overlaid on a medical image. Ideally, the high-dose regions (shown in red) should conform closely to the tumor's shape, with minimal radiation spillover into surrounding healthy tissues. While clinical expertise is required for a detailed interpretation, it is clear that using the RMR sparse matrix, as opposed to the naive sparse matrix, results in reduced radiation exposure to the right lung (visible on the left side of the figure).

### 4.3   Limitations and Broader Impacts

We acknowledge that this study primarily focused on leveraging advanced matrix sparsification techniques to accurately and efficiently solve the computationally intensive optimization problems in cancer radiotherapy. While we have presented dose maps and DVH figures to illustrate potential improvements in treatment plan quality, our analysis remains limited. Future studies are necessary to comprehensively evaluate the clinical benefits of our techniques. Additionally, this study was reviewed by machine learning experts who focused on the technical aspects rather than the clinical implications of the study. Another limitation of this study is that the theoretical bounds for feasibility and optimality gaps are provided for the quadratic optimization problem 2, not the general problem 1. It can be proven that, by assuming Lipschitz continuity for the functions $f_0$ and $g$, a small $||A - S||_2$ ensures small feasibility and optimality gaps. However, the bounds will depend on the Lipschitz constants of $f_0$ and $g$.

Although our work primarily focused on Intensity Modulated Radiation Therapy (IMRT) with photon radiation, a widely used treatment modality, our approach could have a much broader impact in the field of radiotherapy. This is because all treatment modalities eventually boil down to solving constrained optimization problems with large and dense matrices. As technology advances and new digital machines with greater flexibility, such as those allowing couch movement, become available, the resultant optimization problems are becoming larger, making efficient approximation techniques increasingly essential to solve these problems within clinical timeframes [9, 11].

## 5   Conclusion

We introduced a novel and high-impact application of matrix sparsification alongside an innovative algorithm that combines desirable theoretical guarantees with superior experimental performance. Our algorithm creates a highly sparse, randomized sketch of the original dose influence matrix used in radiotherapy optimization, preserving essential information. This enables solving the large-scale radiotherapy optimization problems within clinically viable timeframes, ensuring minimal discrepancy between the optimized radiation dose and the actual dose received by patients. It effectively tackles the "garbage-in-garbage-out" problem prevalent in current radiotherapy optimization caused by reliance on inaccurately sparsified matrices from the current "naive" approaches. This method holds significant promise for enhancing the quality of radiotherapy treatments.

## Acknowledgments and Disclosure of Funding

Gourav Jhanwar and Masoud Zarepisheh gratefully acknowledge funding from the MSK Cancer Center Support Grant/Core Grant from the NIH (P30 CA008748).

## Footnotes

*Correspondence to `mtefagh@sharif.edu`

## References

[1] Dimitris Achlioptas and Frank McSherry. Fast computation of low rank matrix approximations. In *Proceedings of the Thirty-Third Annual ACM Symposium on Theory of Computing*, STOC '01, page 611–618, New York, NY, USA, 2001. Association for Computing Machinery.

[2] Dimitris Achlioptas, Zohar S Karnin, and Edo Liberty. Near-optimal entrywise sampling for data matrices. In *Advances in Neural Information Processing Systems*, volume 26, pages 1565–1573. Curran Associates, Inc., 2013.

[3] Sanjeev Arora, Elad Hazan, and Satyen Kale. A fast random sampling algorithm for sparsifying matrices. In *Proceedings of the 9th International Conference on Approximation Algorithms for Combinatorial Optimization Problems, and 10th International Conference on Randomization and Computation*, page 272–279. Springer-Verlag, 2006.

[4] Vladimir Braverman, Robert Krauthgamer, Aditya R. Krishnan, and Shay Sapir. Near-optimal entrywise sampling of numerically sparse matrices. In Mikhail Belkin and Samory Kpotufe, editors, *Proceedings of Thirty Fourth Conference on Learning Theory*, volume 134 of *Proceedings of Machine Learning Research*, pages 759–773. PMLR, 15–19 Aug 2021.

[5] Steven Diamond and Stephen Boyd. Cvxpy: A python-embedded modeling language for convex optimization. *Journal of Machine Learning Research*, 17(83):1–5, 2016.

[6] Felix C Difilippo. Forward and adjoint methods for radiotherapy planning. *Medical Physics*, 25 (9):1702–1710, 1998.

[7] Petros Drineas and Anastasios Zouzias. A note on element-wise matrix sparsification via a matrix-valued bernstein inequality. *Inf. Process. Lett.*, 111(8):385–389, 2011.

[8] Alex Gittens and Joel A Tropp. Error bounds for random matrix approximation schemes. *arXiv preprint arXiv:0911.4108*, 2009.

[9] Charles Huang, Yong Yang, and Lei Xing. Fully automated noncoplanar radiation therapy treatment planning. *Medical Physics*, 48(11):7439–7449, 2021.

[10] Gourav Jhanwar, Mojtaba Tefagh, Vicki T Taasti, Sadegh R Alam, Seppo Tuomaala, Saad Nadeem, and Masoud Zarepisheh. Portpy: An open-source python package for planning and optimization in radiation therapy including benchmark data and algorithms. *AAPM 65th Annual Meeting & Exhibition*, 2023.

[11] Angelia Landers, Daniel O'Connor, Dan Ruan, and Ke Sheng. Automated $4\pi$ radiotherapy treatment planning with evolving knowledge-base. *Medical physics*, 46(9):3833–3843, 2019.

[12] H. Liu, D. Schaal, H. Curry, R. Clark, A. Magliari, P. Kupelian, D. Khuntia, and S. Beriwal. Review of cone beam computed tomography based online adaptive radiotherapy: current trend and future direction. *Radiation oncology (London, England)*, 18(1), 2023.

[13] ApS MOSEK. Mosek optimizer api for python. *Software Package, Ver*, 9, 2020.

[14] Riley Murray, James Demmel, Michael W Mahoney, N Benjamin Erichson, Maksim Melnichenko, Osman Asif Malik, Laura Grigori, Piotr Luszczek, Michał Dereziński, Miles E Lopes, et al. Randomized numerical linear algebra: A perspective on the field with an eye to software. *arXiv preprint arXiv:2302.11474*, 2023.

[15] NH Nguyen, Petros Drineas, and TD Tran. Matrix sparsification via the khintchine inequality. 2009.

[16] Mert Pilanci and Martin J Wainwright. Iterative hessian sketch: Fast and accurate solution approximation for constrained least-squares. *Journal of Machine Learning Research*, 17(53): 1–38, 2016.

[17] H Edwin Romeijn, James F Dempsey, and Jonathan G Li. A unifying framework for multi-criteria fluence map optimization models. *Physics in Medicine & Biology*, 49(10):1991, 2004.

[18] H. Edwin Romeijn, Ravindra K. Ahuja, James F. Dempsey, and Arvind Kumar. A column generation approach to radiation therapy treatment planning using aperture modulation. *SIAM Journal on Optimization*, 15(3):838–862, 2005.

[19] David M. Shepard, Michael C. Ferris, Gustavo H. Olivera, and T. Rockwell Mackie. Optimizing the delivery of radiation therapy to cancer patients. *SIAM Review*, 41(4):721–744, 1999.

[20] Joel A. Tropp. User-friendly tail bounds for sums of random matrices. *Foundations of Computational Mathematics*, 12(4):389–434, 2011.

[21] David P Woodruff et al. Sketching as a tool for numerical linear algebra. *Foundations and Trends® in Theoretical Computer Science*, 10(1–2):1–157, 2014.

[22] M. Zarepisheh, L. Hong, Y. Zhou, Q. Huang, J. Yang, G. Jhanwar, H. D. Pham, P. Dursun, P. Zhang, M. A. Hunt, G. S. Mageras, J. T. Yang, Y. Yamada, and J. O. Deasy. Automated and clinically optimal treatment planning for cancer radiotherapy. *INFORMS journal on applied analytics*, 52(1):69–89, 2022.

# A Proofs

## A.1 Proof of Lemma 3.1

*Proof.* The proof proceeds through straightforward analysis. Consider that for each row $i$ of matrix $A$, the inequality $\Sigma_i = \sum_{j \in T_i} |a_{ij}| \leq \epsilon |T_i|$ holds. Consequently, the number of samples $k_i$ for row $i$ satisfies $k_i \leq |T_i|$. Furthermore, $|T_i|$ is at most the count of non-zero entries in the $i$-th row of $A$. Summing over all rows, the total number of samples is indeed less than or equal to $\mathrm{nnz}(A)$. $\qquad\square$

## A.2 Proof of Theorem 3.2

We employ the following theorem and lemmas to prove Theorem 3.2. The initial theorem establishes a probabilistic bound for the $\ell_2$ norm of the aggregate of random matrices, serving as an essential instrument in various contexts. This theorem has also been utilized by Achlioptas et al. [2] and Drineas and Zouzias [7] to establish their theoretical guarantees.

**Theorem A.1** (Matrix Bernstein [20]). *Suppose that $X_1, X_2, \ldots X_t$ are independent, zero mean, random matrices with dimensions $m \times n$. If for each $1 \leq i \leq t$, $\|X_i\|_2 \leq R$ almost surely and*

$$\max\left(\left\|\sum_{i=1}^{t} \mathbb{E}\left[X_i X_i^\top\right]\right\|_2, \left\|\sum_{i=1}^{t} \mathbb{E}\left[X_i^\top X_i\right]\right\|_2\right) \leq \sigma^2,$$

*then for all $\tau \geq 0$,*

$$\mathbb{P}\left\{\left\|\sum_{i=1}^{t} X_i\right\|_2 \geq \tau\right\} \leq (m+n)\exp\left(\frac{-\frac{\tau^2}{2}}{\sigma^2 + \frac{R\tau}{3}}\right).$$

**Lemma A.2.** *Suppose that $v = \begin{bmatrix} v_1 & v_2 & \cdots & v_n \end{bmatrix}^\top$ is a vector. Then the matrix $M = \|v\|_1 \mathrm{diag}(|v_1|, |v_2|, \ldots, |v_n|) - vv^\top$ is a positive semi-definite matrix.*

*Proof.* It is sufficient to prove that for every vector $x = \begin{bmatrix} x_1 & x_2 & \cdots & x_n \end{bmatrix}^\top$, $x^\top M x \geq 0$. Notice that

$$x^\top M x = \|v\|_1 \sum_{i=1}^{n} |v_i| x_i^2 - \left(x^\top v\right)^2 = \left(\sum_{i=1}^{n} |v_i|\right)\left(\sum_{i=1}^{n} |v_i| x_i^2\right) - \left(\sum_{i=1}^{n} v_i x_i\right)^2 \geq 0.$$

The last inequality is a result of Cauchy–Schwarz inequality because we can set $a_i = \sqrt{|v_i|}$ for $v_i \geq 0$, and $a_i = -\sqrt{|v_i|}$ for $v_i \leq 0$ and $b_i = \sqrt{|v_i|} x_i$, in the below Cauchy-Schwarz inequality:

$$\left(\sum_{i=1}^{n} a_i b_i\right)^2 \leq \left(\sum_{i=1}^{n} a_i^2\right)\left(\sum_{i=1}^{n} b_i^2\right).$$

$\qquad\square$

**Lemma A.3.** *Let $\alpha_1, \alpha_2, \ldots, \alpha_m$ be positive real numbers, $v_1, v_2, \ldots, v_m \in \mathbb{R}^n$ and $M \in \mathbb{R}^{n \times n}$. If $M' = M - \sum_{i=1}^{m} \alpha_i v_i v_i^\top$ is a positive semi-definite matrix then $\|M'\|_2 \leq \|M\|_2$.*

*Proof.* Since $M'$ is a positive semi-definite matrix, $\|M'\|_2 = \sup_{\|x\|_2=1} x^\top M' x$. Note that

$$x^\top M' x = x^\top M x - \sum_{i=1}^{m} \alpha_i \left(x^\top v_i\right)^2 \leq |x^\top M x| \leq \|M\|_2,$$

since $M$ is a symmetric matrix. Hence $\|M'\|_2 \leq \|M\|_2$. $\qquad\square$

Now, we offer a proof for Theorem 3.2.

*Proof.* Let $M_i = \sum_{j=1}^n |a_{ij}|$ and define $E_i = \{|a_{ij}| \mid 1 \leq j \leq n, |a_{ij}| > \epsilon\}$. As per Algorithm 1, the sum of the elements in $E_i$ equals $M_i - \Sigma_i$. Consequently, the cardinality of $E_i$, denoted by $|E_i|$, is at most $(M_i - \Sigma_i)/\epsilon$. Additionally, the number of samples is $\lceil \Sigma_i/\epsilon \rceil$. Therefore, the number of nonzero entries in the $i$-th row is less than $M_i/\epsilon + 1$. Consequently, the total number of non-zero (nnz) entries in $S$ is less than $m + \sum_{i=1}^m M_i/\epsilon = m + \|A\|_1 / \epsilon$.

Assume that in the $i$-th row and during the $t$-th iteration of the redistribution step in Algorithm 1, index $j$ is selected. Let $S_{it}$ represent an $m \times n$ matrix characterized by a single non-zero entry, $\text{sign}(a_{ij}) \cdot \Sigma_i$, located in the $i$-th row and $j$-th column. Now, let us zero out all entries of $A$ that are greater than $\epsilon$ in absolute value and call the new matrix $\hat{A}$, and define $r_i^\top$ the $i$-th row of $\hat{A}$. It becomes apparent that $\Sigma_i = \|r_i\|_1$. Consequently, we have

$$S = A - \hat{A} + \sum_{i=1}^m \frac{1}{k_i} \sum_{t=1}^{k_i} S_{it}.$$

Then

$$A - S = \sum_{i=1}^m \left( \hat{A}_{(i)} - \frac{1}{k_i} \sum_{t=1}^{k_i} S_{it} \right) = \sum_{i=1}^m \left( \frac{1}{k_i} \sum_{t=1}^{k_i} \hat{A}_{(i)} - \frac{1}{k_i} \sum_{t=1}^{k_i} S_{it} \right) = \sum_{i=1}^m \sum_{t=1}^{k_i} \frac{1}{k_i} \left( \hat{A}_{(i)} - S_{it} \right),$$

and given that

$$\mathbb{E}[S_{it}]_{ij} = \frac{|\hat{a}_{ij}|}{\Sigma_i} \text{sign}(\hat{a}_{ij}) \Sigma_i = \hat{a}_{ij},$$

we have $\mathbb{E}[S_{it}] = \hat{A}_{(i)}$. Now denote $(\hat{A}_{(i)} - S_{it})/k_i$ by $X_{it}$. Since $X_{it}$ has just one nonzero row, $X_{it} X_{it}^\top$ has just one nonzero entry which is on its diagonal. This entry is equal to

$$\frac{1}{k_i^2} \left( (a_{ij} - \Sigma_i \text{sign}(a_{ij}))^2 + \sum_{l \neq j}^n a_{il}^2 \right) = \frac{1}{k_i^2} \left( \|r_i\|_2^2 - 2|a_{ij}|\Sigma_i + \Sigma_i^2 \right),$$

for some $1 \leq j \leq n$. Therefore $\|X_{it}\|_2 = \sqrt{(\|r_i\|_2^2 - 2|a_{ij}|\Sigma_i + \Sigma_i^2)/k_i^2}$. Also

$$\sqrt{\frac{1}{k_i^2} \left( \|r_i\|_2^2 - 2|a_{ij}|\Sigma_i + \Sigma_i^2 \right)} \leq \sqrt{\frac{\epsilon^2}{\Sigma_i^2} (2\Sigma_i^2)} = \sqrt{2}\epsilon,$$

and due to Theorem A.1 one can let $R = \sqrt{2}\epsilon$.
Note that

$$\sum_{i=1}^m \sum_{t=1}^{k_i} \mathbb{E}\left[ X_{it}^\top X_{it} \right] = \sum_{i=1}^m k_i \mathbb{E}\left[ X_{i1}^\top X_{i1} \right],$$

so we just need to calculate $k_i \mathbb{E}\left[ X_{i1}^\top X_{i1} \right]$, which is equal to

$$k_i \mathbb{E}\left[ \frac{1}{k_i^2} \left( \hat{A}_{(i)}^\top - S_{i1}^\top \right) \left( \hat{A}_{(i)} - S_{i1} \right) \right] = \frac{1}{k_i} \left( \hat{A}_{(i)}^\top \hat{A}_{(i)} - \hat{A}_{(i)}^\top \mathbb{E}[S_{i1}] - \mathbb{E}[S_{i1}^\top] \hat{A}_{(i)} + \mathbb{E}[S_{i1}^\top S_{i1}] \right)$$

$$= \frac{1}{k_i} \left( -\hat{A}_{(i)}^\top \hat{A}_{(i)} + \mathbb{E}[S_{i1}^\top S_{i1}] \right)$$

$$= \frac{1}{k_i} \left( -r_i r_i^\top + \mathbb{E}[S_{i1}^\top S_{i1}] \right).$$

Recall that $S_{i1}$ has just one nonzero entry, therefore $S_{i1}^\top S_{i1}$ is a diagonal matrix. Also

$$\mathbb{E}\left[ S_{i1}^\top S_{i1} \right]_{jj} = \frac{|\hat{a}_{ij}|}{\Sigma_i} \Sigma_i^2 = |\hat{a}_{ij}| \Sigma_i.$$

From Lemma A.2, one can deduce that $\sum_{i=1}^m \sum_{t=1}^{k_i} \mathbb{E}\left[ X_{it}^\top X_{it} \right]$ is a positive semi-definite matrix and from Lemma A.3, its spectral norm is less than or equal to the spectral norm of $\sum_{i=1}^m \mathbb{E}\left[ S_{i1}^\top S_{i1} \right]/k_i$, which is a diagonal matrix and

$$\left( \sum_{i=1}^m \frac{1}{k_i} \mathbb{E}\left[ S_{i1}^\top S_{i1} \right] \right)_{jj} = \sum_{i=1}^m \frac{|\hat{a}_{ij}|\Sigma_i}{k_i} \leq \sum_{i=1}^m |\hat{a}_{ij}|\epsilon \leq m\epsilon^2.$$

Hence $\left\|\sum_{i=1}^{m}\sum_{t=1}^{k_i}\mathbb{E}\left[X_{it}^{\top}X_{it}\right]\right\|_2 \leq m\epsilon^2$.

According to Theorem A.1, there should also be an upper bound for $\left\|\sum_{i=1}^{m}\sum_{t=1}^{k_i}\mathbb{E}\left[X_{it}X_{it}^{\top}\right]\right\|_2$. Similarly,

$$k_i\mathbb{E}\left[X_{i1}X_{i1}^{\top}\right] = \frac{1}{k_i}\left(-\hat{A}_{(i)}\hat{A}_{(i)}^{\top} + \mathbb{E}\left[S_{i1}S_{i1}^{\top}\right]\right).$$

The resulting matrix consists of just a nonzero entry, the $ii-$th entry of which is equal to $(-\|r_i\|_2^2 + \Sigma_i^2)/k_i \geq 0$. Therefore $\sum_{i=1}^{m}\sum_{t=1}^{k_i}\mathbb{E}\left[X_{it}X_{it}^{\top}\right]$ is a diagonal matrix. Then

$$\left\|\sum_{i=1}^{m}\sum_{t=1}^{k_i}\mathbb{E}\left[X_{it}X_{it}^{\top}\right]\right\|_2 = \max_{1\leq i\leq n}\left(\frac{1}{k_i}\left(-\|r_i\|_2^2 + \Sigma_i^2\right)\right)$$

$$\leq \max_{1\leq i\leq n}\left(\frac{\epsilon}{\Sigma_i}\left(-\frac{\Sigma_i^2}{n} + \Sigma_i^2\right)\right)$$

$$= \max_{1\leq i\leq n}\left(\frac{(n-1)\epsilon}{n}\Sigma_i\right)$$

$$\leq (n-1)\epsilon^2.$$

So one can let $\sigma^2 = \max(m, n-1)\epsilon^2$.

Building on Theorem A.1, the derivation of Theorem 3.2 becomes a straightforward computation. We need to have

$$\delta = (m+n)\exp\left(\frac{-\frac{\tau^2}{2}}{\max(m, n-1)\epsilon^2 + \frac{\sqrt{2}\epsilon\tau}{3}}\right).$$

Thus

$$\log\left(\frac{m+n}{\delta}\right)\max(m, n-1)\epsilon^2 + \log\left(\frac{m+n}{\delta}\right)\frac{\sqrt{2}\tau}{3}\epsilon - \frac{\tau^2}{2} = 0.$$

The above equation has just one non-negative root. Therefore

$$\epsilon = \frac{\sqrt{\log^2\left(\frac{m+n}{\delta}\right)\frac{2\tau^2}{9} + 2\log\left(\frac{m+n}{\delta}\right)\max(m, n-1)\tau^2} - \log\left(\frac{m+n}{\delta}\right)\frac{\sqrt{2}\tau}{3}}{2\log\left(\frac{m+n}{\delta}\right)\max(m, n-1)}$$

$$= \frac{\tau\left(\sqrt{1 + \frac{9\max(m, n-1)}{\log\left(\frac{m+n}{\delta}\right)}} - 1\right)}{3\sqrt{2}\max(m, n-1)}.$$

$\square$

## A.3 Proof of Corollary 3.3

*Proof.* Notice that $\sqrt{1+c} - 1 \geq \sqrt{c}/2$ for every $c \geq 16/9$. In Theorem 3.2 let $\delta = 1/(m+n)$. Then we have

$$\frac{\tau\left(\sqrt{1 + \frac{9\max(m, n-1)}{\log\left(\frac{m+n}{\delta}\right)}} - 1\right)}{3\sqrt{2}\max(m, n-1)} = \frac{\tau\left(\sqrt{1 + \frac{9\max(m, n-1)}{2\log(m+n)}} - 1\right)}{3\sqrt{2}\max(m, n-1)} \geq \frac{\frac{\tau}{2}\sqrt{\frac{9\max(m, n-1)}{2\log(m+n)}}}{3\sqrt{2}\max(m, n-1)}$$

$$= \frac{\tau}{4\sqrt{\max(m, n-1)\log(m+n)}},$$

since $9\max(m, n-1)/(2\log(m+n)) \geq 16/9$. Therefore based on Theorem 3.2 the result follows. $\square$

## A.4 Proof of Lemma 3.4

We apply the Bernstein inequality to establish the result.

**Theorem A.4** (Bernstein inequality). *Let $X_1, \ldots, X_n$ be independent zero-mean random variables. Suppose that $|X_i| \leq M$ almost surely, for all $i$. Then, for all positive $\tau$,*

$$\mathbb{P}\left\{\left|\sum_{i=1}^n X_i\right| \geq \tau\right\} \leq 2\exp\left(\frac{-\frac{\tau^2}{2}}{\sum_{i=1}^n \mathbb{E}\left[X_i^2\right] + \frac{M\tau}{3}}\right).$$

Now we present a proof for Lemma 3.4.

*Proof.* Analogous to the proof of Theorem 3.2, we define the $X_{it}$ matrices. Let $y_t^\top$ be the $i$-th row of $X_{it}$. From the proof of Theorem 3.2, the $i$-th row of $A - S$ is given by $\sum_{t=1}^{k_i} y_t^\top$ and $\mathbb{E}\left[y_t^\top\right] = 0$. Then, for each $x \in \mathbb{R}^n$, we have $\mathbb{E}\left[y_t^\top x\right] = 0$ and

$$\mathbb{E}\left[\left(y_t^\top x\right)^2\right] = \mathbb{E}\left[x^\top y_t y_t^\top x\right] = \mathbb{E}\left[\text{tr}\left(x^\top y_t y_t^\top x\right)\right] = \mathbb{E}\left[\text{tr}\left(xx^\top y_t y_t^\top\right)\right]$$
$$= \text{tr}\left(\mathbb{E}\left[xx^\top y_t y_t^\top\right]\right)$$
$$= \text{tr}\left(xx^\top \mathbb{E}\left[y_t y_t^\top\right]\right).$$

Based on the proof of Theorem 3.2, we know that $\mathbb{E}\left[y_t y_t^\top\right] = (-r_i r_i^\top + Y)/k_i^2$, where $Y$ is a diagonal matrix and the $jj$-th entry of it equals $|\hat{a}_{ij}|\Sigma_i$. Hence,

$$\mathbb{E}\left[\left(y_t^\top x\right)^2\right] = \frac{1}{k_i^2}\text{tr}\left(-xx^\top r_i r_i^\top + xx^\top Y\right)$$
$$= \frac{1}{k_i^2}\text{tr}\left(-x^\top r_i r_i^\top x + x^\top Y x\right)$$
$$= \frac{1}{k_i^2}\left(-(r_i^\top x)^2 + \sum_{j=1}^n |\hat{a}_{ij}|\Sigma_i x_j^2\right).$$

Therefore,

$$\mathbb{E}\left[\sum_{t=1}^{k_i}\left(y_t^\top x\right)^2\right] = \frac{1}{k_i}\left(-(r_i^\top x)^2 + \sum_{j=1}^n |\hat{a}_{ij}|\Sigma_i x_j^2\right)$$
$$= \frac{1}{k_i}\left(-\sum_{j=1}^n \hat{a}_{ij}^2 x_j^2 - 2\sum_{1 \leq j < l \leq n} \hat{a}_j \hat{a}_l x_j x_l + \sum_{j=1}^n \hat{a}_{ij}^2 x_j^2 + \sum_{1 \leq j < l \leq n} |\hat{a}_{ij}\hat{a}_{il}|(x_j^2 + x_l^2)\right)$$
$$= \frac{1}{k_i}\sum_{1 \leq j < l \leq n} |\hat{a}_{ij}\hat{a}_{il}|(x_j - \text{sign}(\hat{a}_{ij}\hat{a}_{il})x_l)^2$$
$$\leq \frac{\epsilon}{\Sigma_i}\sum_{1 \leq j < l \leq n} |\hat{a}_{ij}\hat{a}_{il}|2(x_j^2 + x_l^2)$$
$$= \frac{2\epsilon}{\Sigma_i}\sum_{j=1}^n |\hat{a}_{ij}|x_j^2 \sum_{l \neq j}^n |\hat{a}_{il}|$$
$$\leq 2\epsilon^2 \|x\|_2^2.$$

Also, $|y_t^\top x| \leq \|y_t\|_2 \|x\|_2$ and

$$\|y_t\|_2 = \sqrt{\frac{1}{k_i^2}\left((a_{ij} - \Sigma_i\text{sign}(a_{ij}))^2 + \sum_{l \neq j}^n a_{il}^2\right)} = \sqrt{\frac{1}{k_i^2}\left(\|r_i\|_2^2 - 2|a_{ij}|\Sigma_i + \Sigma_i^2\right)}$$
$$\leq \sqrt{\frac{\epsilon^2}{\Sigma_i^2}(2\Sigma_i^2)}$$
$$= \sqrt{2}\epsilon.$$

Now according to Theorem A.4 we have

$$\mathbb{P}\left\{\left|\sum_{t=1}^{k_i} y_t^\top x\right| \geq \tau\right\} \leq 2\exp\left(\frac{-\frac{\tau^2}{2}}{2\epsilon^2 \|x\|_2^2 + \frac{\sqrt{2}\epsilon\|x\|_2\tau}{3}}\right).$$

Setting $\tau = c\epsilon \|x\|_2$ results in

$$2\exp\left(\frac{-\frac{\tau^2}{2}}{2\epsilon^2 \|x\|_2^2 + \frac{\sqrt{2}\epsilon\|x\|_2\tau}{3}}\right) = 2\exp\left(\frac{-c^2}{4 + \frac{2\sqrt{2}c}{3}}\right),$$

thereby completing the proof. $\qquad\square$

## A.5 Proof of Lemma 3.5

*Proof.* Let $d_A$ and $d_S$ denote $Ax$ and $Sx$, respectively. In Lemma 3.4 if we let $c = 19 + 5\log m$, then we have

$$2\exp\left(\frac{-c^2}{4 + \frac{2\sqrt{2}c}{3}}\right) \leq 2\exp\left(\frac{-c^2}{4c + \frac{2\sqrt{2}c}{3}}\right) = 2\exp\left(\frac{-c}{4 + \frac{2\sqrt{2}}{3}}\right)$$

$$= 2\exp\left(\frac{-19 - 5\log m}{4 + \frac{2\sqrt{2}}{3}}\right)$$

$$< \frac{0.05}{m}.$$

Thus

$$\mathbb{P}\left(|(d_A - d_S)_i| \geq (19 + 5\log m)\epsilon\|x\|_2\right) < \frac{0.05}{m}.$$

Applying the union bound, with probability $1 - m \cdot \frac{0.05}{m} = 0.95$, we have

$$\|d_A - d_S\|_\infty < (19 + 5\log m)\epsilon\|x\|_2.$$

Now, if $c^T$ is any arbitrary row of $C$, we have $\|c\|_1 \leq 1$. Therefore,

$$|c^T(d_A - d_S)| \leq \sum_{i=1}^m |c_i||(d_A - d_S)_i| \leq \sum_{i=1}^m |c_i|\|d_A - d_S\|_\infty \leq \|d_A - d_S\|_\infty,$$

which proves the result. $\qquad\square$

## A.6 Proof of Lemma 3.7

*Proof.* Consider

$$f_0(Ax) = \sum_{s\in\bar{S}} \sum_{i\in I_s} \left(w_+^s \max(A_i^s x - d^s, 0)^2 + w_-^s \max(d^s - A_i^s x, 0)^2\right),$$

which is the term dependent on $Ax$ in the objective function of the optimization problem (2). Note that the remaining part of the objective function is independent of $A$ and hence is unaffected by the approximation.

Let $e = (19 + 5\log m)\epsilon\|x\|_2$. For all $s \in \bar{S}$ and $i \in I_s$, we have:

$$\max(S_i^s x - d^s, 0) \geq \max(A_i^s x - e - d^s, 0) \geq \max(A_i^s x - d^s, 0) - e.$$

Similarly, we have:

$$\max(S_i^s x - d^s, 0) \leq \max(A_i^s x + e - d^s, 0) \leq \max(A_i^s x - d^s, 0) + e.$$

Therefore,

$$|\max(A_i^s x - d^s, 0) - \max(S_i^s x - d^s, 0)| \leq e,$$

and,

$$|\max(A_i^s x - d^s, 0) + \max(S_i^s x - d^s, 0)| \leq 2\max(A_i^s x - d^s, 0) + e.$$

Altogether, we have that

$$|\max(A_i^s x - d^s, 0)^2 - \max(S_i^s x - d^s, 0)^2| \le e(2\max(A_i^s x - d^s, 0) + e).$$

By a similar argument, we have that

$$|\max(d^s - A_i^s x, 0)^2 - \max(d^s - S_i^s x, 0)^2| \le e(2\max(d^s - A_i^s x, 0) + e).$$

In conclusion,

$$
\begin{aligned}
|f_0(Ax) - f_0(Sx)| &\le \sum_{s \in \bar{S}, i \in I_s} e\left(w_+^s(2\max(A_i^s x - d^s, 0) + e) + w_-^s(2\max(d^s - A_i^s x, 0) + e)\right) \\
&\le \sum_{s \in \bar{S}, i \in I_s} e\left(w_+^s(\max(A_i^s x - d^s, 0)^2 + 1 + e) + w_-^s(\max(d^s - A_i^s x, 0)^2 + 1 + e)\right) \\
&\le e\left(f_0(Ax) + \sum_{s \in \bar{S}, i \in I_s}(w_+^s + w_-^s)(1 + e)\right) \\
&\le e\left(f_0(Ax) + m(1 + e)\sum_{s \in \bar{S}}(w_+^s + w_-^s)\right).
\end{aligned}
$$

Thus, in summary, for any arbitrary $x \in \mathbb{R}^n$, we have that

$$|f_0(Ax) - f_0(Sx)| \le e\left(f_0(Ax) + m(1 + e)\sum_{s \in \bar{S}}(w_+^s + w_-^s)\right),$$

with probability at least $0.95$. □

## A.7 Proof of Lemma 3.8

*Proof.* It suffices to consider:

$$\hat{x} = \frac{e}{s}\tilde{x} + \left(1 - \frac{e}{s}\right)x.$$

For each convex constraint $g_i \le 0$,

$$
\begin{aligned}
g_i(A\hat{x}) &= g_i\left(A\left(\frac{e}{s}\tilde{x} + (1 - \frac{e}{s})x\right)\right) \\
&\le \frac{e}{s}g_i(A\tilde{x}) + \left(1 - \frac{e}{s}\right)g_i(Ax) \\
&\le \frac{e}{s}(-s) + \left(1 - \frac{e}{s}\right)e \\
&\le -\frac{e^2}{s} \\
&\le 0.
\end{aligned}
$$

Additionally,

$$
\begin{aligned}
\|x - \hat{x}\| &= \left\|x - \left(\frac{e}{s}\tilde{x} + \left(1 - \frac{e}{s}\right)x\right)\right\| \\
&= \left\|\frac{e}{s}(x - \tilde{x})\right\| \\
&= \frac{e}{s}\|x - \tilde{x}\| \\
&\le \frac{e}{s}(\|x\| + \|\tilde{x}\|).
\end{aligned}
$$

This completes the proof. □

## A.8 Proof of Theorem 3.9

*Proof.* Let $s = \min(\min_s(d_{\text{Max}}^s), \min_s(d_{\text{Mean}}^s))$. Then, $\tilde{x} = 0$ satisfies the assumptions of Lemma (3.8) for the approximated optimization problem. Therefore, with a probability of 0.95, there exists a

feasible point of the approximated optimization problem $\hat{x}_A \in \mathbb{R}^n$ such that $\|x_A - \hat{x}_A\| \leq \frac{e}{s}\|x_A\|$.

$$f_0(Ax_S) + f_1(x_S) \leq f_0(Sx_S) + f_1(x_S) + e\big(f_0(Ax_S) + m(1+e)\sum_{s \in \bar{S}}(w_+^s + w_-^s)\big)$$

$$\leq f_0(S\hat{x}_A) + f_1(\hat{x}_A) + e\big(f_0(Ax_S) + m(1+e)\sum_{s \in \bar{S}}(w_+^s + w_-^s)\big)$$

$$\leq f_0(A\hat{x}_A) + f_1(\hat{x}_A) + e\big(f_0(Ax_S) + m(1+e)\sum_{s \in \bar{S}}(w_+^s + w_-^s)\big)$$

$$+ e\big(f_0(A\hat{x}_A) + m(1+e)\sum_{s \in \bar{S}}(w_+^s + w_-^s)\big)$$

$$\leq f_0\left(A(x_A + O(e))\right) + f_1(x_A + O(e)) + O(e)$$

$$\leq f_0\left(Ax_A\right) + f_1(x_A) + O(e)$$

The first and third lines follow from Lemma (3.7), the second line follows from the optimality of $x_S$ for the approximated optimization problem, and the forth line follows from $\frac{e}{s}\|x_A\| = O(e)$. Finally, the last line follows from the Lipschitz continuity of the objective function over the feasible set which is bounded. Altogether, the sub-optimality of the solution to the approximated optimization problem is $O(e)$. $\qquad\square$

## B   Additional Experimental Results

**Dataset and problem formulation.**   Table 2 presents data for ten randomly selected lung patients, detailing the number of rows (i.e., patient's voxels), columns (i.e., machine's beamlets), and the count of non-zero elements. Each patient received a prescribed radiation dose of 60 Gray. The treatment plans involved 9 manually selected radiation beams, chosen by an expert clinical physicist and tailored to each patient's anatomy. Table 3 presents the maximum and mean dose constraints used in the optimization problem 2. All the data have been downloaded from PortPy.

Table 2: Data size summary for the 10 patients in the study.

|  | Number of voxels (matrix rows, $m$) | Number of beamlets (matrix columns, $n$) | Number of non-zeros (nnz($A$)) |
|---|---|---|---|
| 1 | 381,343 | 2,095 | 257,833,896 |
| 2 | 336,384 | 2,397 | 281,798,395 |
| 3 | 438,138 | 1,704 | 211,556,826 |
| 4 | 324,926 | 1,501 | 135,431,085 |
| 5 | 292,120 | 1,185 | 91,472,882 |
| 6 | 365,469 | 1,545 | 170,601,424 |
| 7 | 281,860 | 1,924 | 201,124,333 |
| 8 | 284,296 | 1,074 | 83,910,465 |
| 9 | 354,925 | 1,261 | 102,218,449 |
| 10 | 385,600 | 1,517 | 143,582,667 |

**Optimization Runtime for Full and RMR Sparse Matrices.**   Table 4 provides a comparison between the optimization runtime when using the full matrix and when employing the RMR sparse matrix with 98% sparsity. The optimization using the full matrix was performed on a high-performance computing (HPC) system with approximately 320 GB of memory, while the RMR sparse matrix optimization was conducted on a system with 128 GB of RAM. Despite the more advanced hardware for the full matrix, the results demonstrate significant improvements in optimization time with the RMR sparse matrix, especially in higher-dimensional cases. Additionally, the execution time of the RMR algorithm is reported, indicating the overhead introduced by the sparsification process.

Table 3: Recommended clinical dose bounds for lung cancer patients.

| Structure | Dose constraint (Gray) |
|---|---|
| Tumor (PTV) | max dose $< 69$ |
| Esophagus | max dose $\leq 66$ |
| | mean dose $\leq 34$ |
| Heart | max dose $\leq 66$ |
| | mean dose $\leq 27$ |
| Cord | max dose $\leq 50$ |
| Lungs | max dose $\leq 66$ |

Nevertheless, the total computational cost remains substantially lower compared to the full matrix approach. The primary computational bottleneck in this application stems from the optimization problem, where the computational time increases cubically with respect to the size of the input matrix. In contrast, the computational time for the RMR algorithm grows linearly.

Table 4: Comparison of Optimization Runtime Using the Full Matrix vs. the RMR Sparse Matrix (98% Sparsity) and RMR Algorithm Runtime

| | Full Matrix Optimization Time (s) | RMR Sparse Matrix Optimization Time (s) | RMR Algorithm Execution Time (s) |
|---|---|---|---|
| 1 | 2826 | 87.25 | 48.77 |
| 2 | 3212 | 122.5 | 46.80 |
| 3 | 2147 | 45.84 | 50.30 |
| 4 | 1277 | 53.30 | 32.78 |
| 5 | 870 | 32.17 | 25.33 |
| 6 | 1447 | 58.25 | 38.26 |
| 7 | 1492 | 81.47 | 33.65 |
| 8 | 714 | 29.18 | 23.88 |
| 9 | 838 | 33.20 | 29.95 |
| 10 | 1303 | 51.66 | 36.59 |

**Ten patients, fixed sparsification level.** Table 7 presents the performance of different algorithms in terms of relative $\ell_2$-norm error, algorithm runtime, feasibility gap, and relative optimality gap for ten lung patients with a fixed sparsification level of 98%. This table provides a detailed expansion of the results shown in Figure 2. Better scores are highlighted in bold for easier visual comparison. Standard deviations are provided for all algorithms, except the naive algorithm, which is deterministic. As can be readily seen, the AHK06 algorithm excels in terms of $\ell_2$-norm error, while the RMR algorithm consistently outperforms in terms of feasibility and optimality gaps. The naive algorithm is the fastest due to its simple implementation.

**Nine patients, various sparsification levels.** Figures 4 to 6 expands the comparisons made for a single patient in Figure 1 to nine additional patients. These figures confirm that the results observed for one patient are consistent across other patients.

**Dose Volume Histogram (DVH) comparisons.** A DVH is a two-dimensional plot used in radiotherapy to depict the distribution of radiation dose within the tumor and the surrounding normal tissues (Figures 7 and 8). Each structure has its own curve on the DVH, where the horizontal axis

represents the dose, and the vertical axis indicates the percentage of the volume receiving at least that dose. For instance, in the top-left figure for the lungs (i.e., left and right lungs), the solid line intersects the point (20,10), signifying that 10% of the lungs receives at least a 10 Gy radiation dose. By illustrating the volume of tissue exposed to various dose levels, DVHs help clinicians assess the uniformity and adequacy of dose coverage to the tumor and ensure that surrounding healthy tissues are spared as much as possible. This visualization is crucial for comparing and optimizing treatment plans, enabling effective tumor control while minimizing adverse effects on normal tissues.

Figures 7 and 8 showcase three DVH plots: the naive approach (left), representing current practice; the AHK06 algorithm (middle), representing the most competitive existing approach; and the proposed RMR algorithm (right). In each plot, dashed lines illustrate the approximated radiation dose, $Sx_S$, used in the optimization problem 2, while solid lines depict the actual radiation dose, $Ax_S$. While clinical expertise is necessary to fully interpret these curves, the gap between the solid and dashed lines indicates the discrepancies resulting from using the approximated matrix $S$ in the optimization problem 2, with a smaller gap being preferred. It is readily apparent that the gap is much smaller, especially for the tumor, when using the RMR algorithm, where the solid and dashed lines are closely aligned and often overlap, making the dashed lines nearly invisible.

**Prostate Patients.** Previously, we focused exclusively on lung patients; we now examine the results for five different prostate patients. As shown in Figures 9 and 10, the results are largely consistent with those for lung patients, with only a few minor differences. The proposed RMR method demonstrates a shorter runtime than AHK06 across all five patients. At high sparsity levels, such as 99%, however, AKL13 outperforms RMR in terms of the relative $\ell_2$-norm metric. Additionally, DZ11 exhibits a higher feasibility gap at lower sparsity levels, indicating that additional samples do not reduce the feasibility gap for this method as expected. Table 5 reveals that the dose influence matrices for prostate patients are considerably denser than those for lung patients, accounting for some observed differences in results. Finally, Figure 11 presents the DVH plots for these five patients, with three DVH plots for each patient: the left plot represents the naive approach, the middle plot shows the AHK06 algorithm, and the right plot displays the RMR algorithm. These findings are consistent with those observed for lung patients, except in the naive approach, which appears to outperform AHK06. This suggests that prior sparsification methods may not consistently outperform the naive approach within this application.

Table 5: Data size summary for the 5 prostate patients.

|   | Number of voxels (matrix rows, $m$) | Number of beamlets (matrix columns, $n$) | Number of non-zeros ($\mathrm{nnz}(A)$) |
|---|---|---|---|
| 1 | 119,587 | 3,555 | 241,300,640 |
| 2 | 130,296 | 2,978 | 243,867,050 |
| 3 | 126,941 | 3,753 | 310,181,024 |
| 4 | 128,867 | 3,848 | 348,771,625 |
| 5 | 132,089 | 3,983 | 355,875,426 |

Table 6: Recommended clinical dose bounds for prostate cancer patients.

| Structure | Dose constraint (Gray) |
|---|---|
| Tumor (PTV) | max dose $\leq 77.2$ |
| Rectum | max dose $\leq 71.6$ |
|  | mean dose $\leq 25$ |
| Bladder | max dose $\leq 75$ |

Table 7: The performance of different algorithms in terms of relative $\ell_2$-norm error, algorithm runtime, feasibility gap, and relative optimality gap for ten lung patients with a fixed sparsification level of 98%.

| | | 1 | 2 | 3 | 4 | 5 | 6 | 7 | 8 | 9 | 10 |
|---|---|---|---|---|---|---|---|---|---|---|---|
| **Relative $\ell_2$-norm (%)** | Naive | 27.22 | 22.24 | 21.62 | 28.08 | 25.47 | 24.11 | 19.2 | 26.18 | 27.38 | 28.18 |
| | AHK06 | **8.64** | 6.77 | **7.55** | **10.09** | **11.23** | **9.06** | 6.48 | **10.78** | **10.82** | **10.2** |
| | | ±0.03 | ±0.01 | ±0.01 | ±0.08 | ±0.02 | ±0.02 | ±0.02 | ±0.02 | ±0.04 | ±0.04 |
| | DZ11 | 14.43 | 11.27 | 13.68 | 16.42 | 18.14 | 15.03 | 10.53 | 16.99 | 17.39 | 17.39 |
| | | ±0.30 | ±0.32 | ±0.25 | ±0.28 | ±0.21 | ±0.07 | ±0.23 | ±0.23 | ±0.17 | ±0.36 |
| | AKL13 | 12.27 | 10.02 | 12.39 | 14.78 | 16.17 | 13.88 | 10.57 | 16.12 | 15.46 | 14.74 |
| | | ±0.02 | ±0.01 | ±0.03 | ±0.01 | ±0.05 | ±0.03 | ±0.02 | ±0.04 | ±0.03 | ±0.03 |
| | BKKS21 | 8.79 | **6.76** | 7.95 | 10.75 | 11.81 | 9.20 | **6.43** | 11.23 | 11.01 | 10.54 |
| | | ±0.02 | ±0.01 | ±0.01 | ±0.05 | ±0.09 | ±0.02 | ±0.01 | ±0.03 | ±0.04 | ±0.02 |
| | RMR | 9.32 | 7.03 | 8.31 | 11.34 | 11.86 | 9.70 | 7.02 | 11.73 | 11.38 | 11.42 |
| | | ±0.05 | ±0.01 | ±0.02 | ±0.01 | ±0.05 | ±0.01 | ±0.02 | ±0.01 | ±0.03 | ±0.07 |
| **Runtime (seconds)** | Naive | **21.84** | **24.7** | **18.31** | **10.42** | **6.11** | **12.30** | **14.05** | **5.24** | **9.59** | **13.23** |
| | AHK06 | 46.91 | 53.33 | 40.19 | 24.02 | 14.73 | 27.94 | 30.69 | 12.63 | 21.90 | 30.06 |
| | | ±0.11 | ±0.19 | ±0.06 | ±0.04 | ±0.04 | ±0.09 | ±0.06 | ±0.04 | ±0.05 | ±0.04 |
| | DZ11 | 3.3e2 | 4.3e2 | 3.3e2 | 1.0e2 | 68.09 | 1.9e2 | 3.2e2 | 51.53 | 84.56 | 1.2e2 |
| | | ±1.7 | ±2.00 | ±2.35 | ±0.77 | ±0.42 | ±0.36 | ±1.00 | ±0.44 | ±0.58 | ±0.95 |
| | AKL13 | 57.51 | 60.71 | 48.54 | 30.54 | 18.99 | 38.10 | 39.88 | 17.06 | 26.42 | 34.13 |
| | | ±0.38 | ±0.52 | ±0.36 | ±0.05 | ±0.12 | ±0.06 | ±0.23 | ±0.08 | ±0.19 | ±0.02 |
| | BKKS21 | 78.79 | 85.07 | 67.96 | 40.93 | 28.28 | 49.28 | 51.35 | 24.36 | 37.80 | 51.13 |
| | | ±0.14 | ±0.24 | ±0.17 | ±0.12 | ±0.26 | ±0.44 | ±0.03 | ±0.07 | ±0.24 | ±0.11 |
| | RMR | 48.77 | 46.80 | 50.30 | 32.78 | 25.33 | 38.26 | 33.65 | 23.88 | 29.95 | 36.59 |
| | | ±0.04 | ±0.04 | ±1.33 | ±0.03 | ±0.03 | ±0.17 | ±0.02 | ±0.19 | ±0.07 | ±0.03 |
| **Feasibility gap** | Naive | 1.3e4 | 6.2e3 | 7.9e3 | 7.6e3 | 2.8e3 | 5.1e3 | 3.6e3 | 4.9e3 | 6.1e3 | 8.2e3 |
| | AHK06 | 0.06 | 0.40 | 0.09 | 0.34 | 0.15 | 0.61 | 0.04 | 0.16 | 0.27 | 0.28 |
| | | ±0.04 | ±0.16 | ±0.05 | ±0.21 | ±0.14 | ±0.26 | ±0.04 | ±0.10 | ±0.16 | ±0.17 |
| | DZ11 | 1.98 | 5.83 | 1.10 | 3.20 | 3.60 | 3.61 | 0.57 | 2.28 | 2.23 | 3.03 |
| | | ±0.64 | ±1.05 | ±0.26 | ±1.55 | ±0.62 | ±0.57 | ±0.10 | ±0.72 | ±1.05 | ±1.44 |
| | AKL13 | 0.48 | 1.03 | 0.16 | 0.49 | 0.99 | 0.97 | 0.21 | 0.26 | 0.25 | 0.30 |
| | | ±0.16 | ±0.25 | ±0.14 | ±0.33 | ±0.48 | ±0.27 | ±0.06 | ±0.30 | ±0.17 | ±0.21 |
| | BKKS21 | 0.02 | **0.26** | 0.07 | 0.05 | 0.04 | **0.43** | **0.03** | 0.06 | **0.08** | 0.07 |
| | | ±0.03 | ±0.09 | ±0.04 | ±0.03 | ±0.04 | ±0.02 | ±0.04 | ±0.06 | ±0.06 | ±0.08 |
| | RMR | **0.00** | 0.39 | **0.05** | **0.04** | **0.00** | 0.45 | 0.04 | **0.00** | 0.10 | **0.01** |
| | | ±0.01 | ±0.08 | ±0.02 | ±0.02 | ±0.00 | ±0.08 | ±0.02 | ±0.00 | ±0.09 | ±0.01 |
| **Optimality gap (%)** | Naive | 1.2e4 | 5.0e3 | 7.2e3 | 1.5e4 | 8.1e3 | 4.3e3 | 3.7e3 | 1.2e4 | 1.4e4 | 1.3e4 |
| | AHK06 | 1.8e2 | 1.2e2 | 1.5e2 | 2.7e2 | 3.0e2 | 1.3e2 | 1.0e2 | 3.3e2 | 3.5e2 | 2.5e2 |
| | | ±3.80 | ±3.69 | ±4.50 | ±5.99 | ±18.6 | ±7.97 | ±0.85 | ±13.4 | ±19.1 | ±7.72 |
| | DZ11 | 3.6e2 | 3.0e2 | 3.0e2 | 4.3e2 | 5.6e2 | 2.2e2 | 2.9e2 | 4.7e2 | 5.3e2 | 4.6e2 |
| | | ±10.8 | ±6.32 | ±6.59 | ±26.8 | ±17.3 | ±6.84 | ±12.1 | ±12.6 | ±32.7 | ±26.4 |
| | AKL13 | 3.5e2 | 3.0e2 | 4.4e2 | 6.8e2 | 1.0e3 | 3.7e2 | 3.0e2 | 9.2e2 | 9.7e2 | 7.4e2 |
| | | ±7.67 | ±10.2 | ±27.7 | ±68.5 | ±72.4 | ±22.7 | ±17.2 | ±51.8 | ±1.2e2 | ±60.2 |
| | BKKS21 | 98.86 | 65.79 | 74.98 | 1.4e2 | 1.4e2 | 54.88 | 62.29 | 1.5e2 | 1.8e2 | 1.4e2 |
| | | ±1.04 | ±0.63 | ±2.52 | ±6.16 | ±5.92 | ±2.91 | ±1.84 | ±3.40 | ±10.9 | ±2.43 |
| | RMR | **18.89** | **13.91** | **11.61** | **25.59** | **21.18** | **9.44** | **11.58** | **23.20** | **32.81** | **26.86** |
| | | ±0.30 | ±0.24 | ±0.38 | ±1.48 | ±1.16 | ±1.75 | ±0.22 | ±0.78 | ±1.63 | ±0.31 |

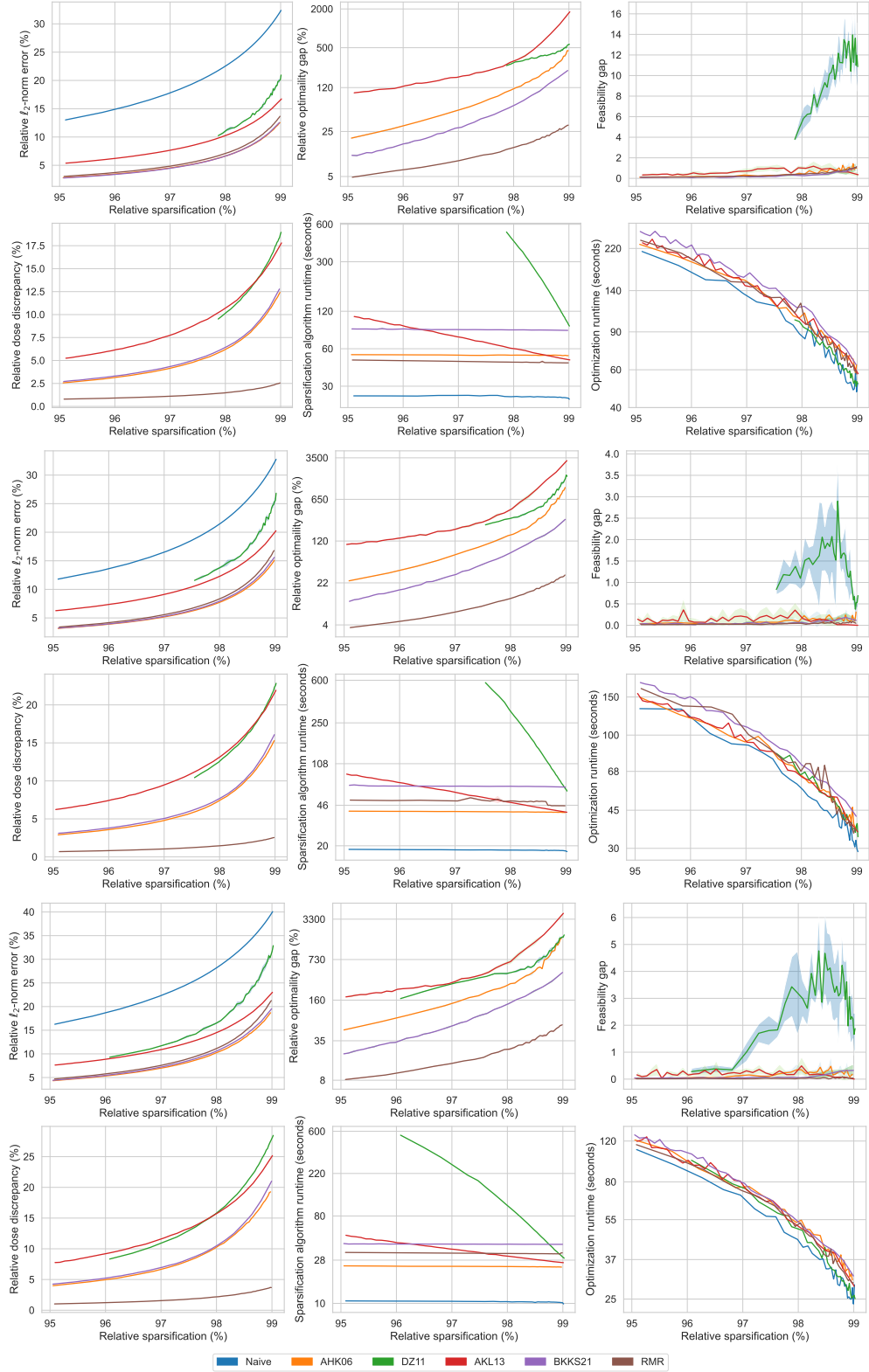

Figure 4: *(Various sparsification levels, Lung patients 2-4)* The performance of different algorithms in terms of: relative $\ell_2$-norm error, relative optimality gap, feasibility gap, relative dose discrepancy, algorithm runtime, and optimization runtime, for patients 2 (first two rows), 3 (second two rows), and 4 (third two rows).

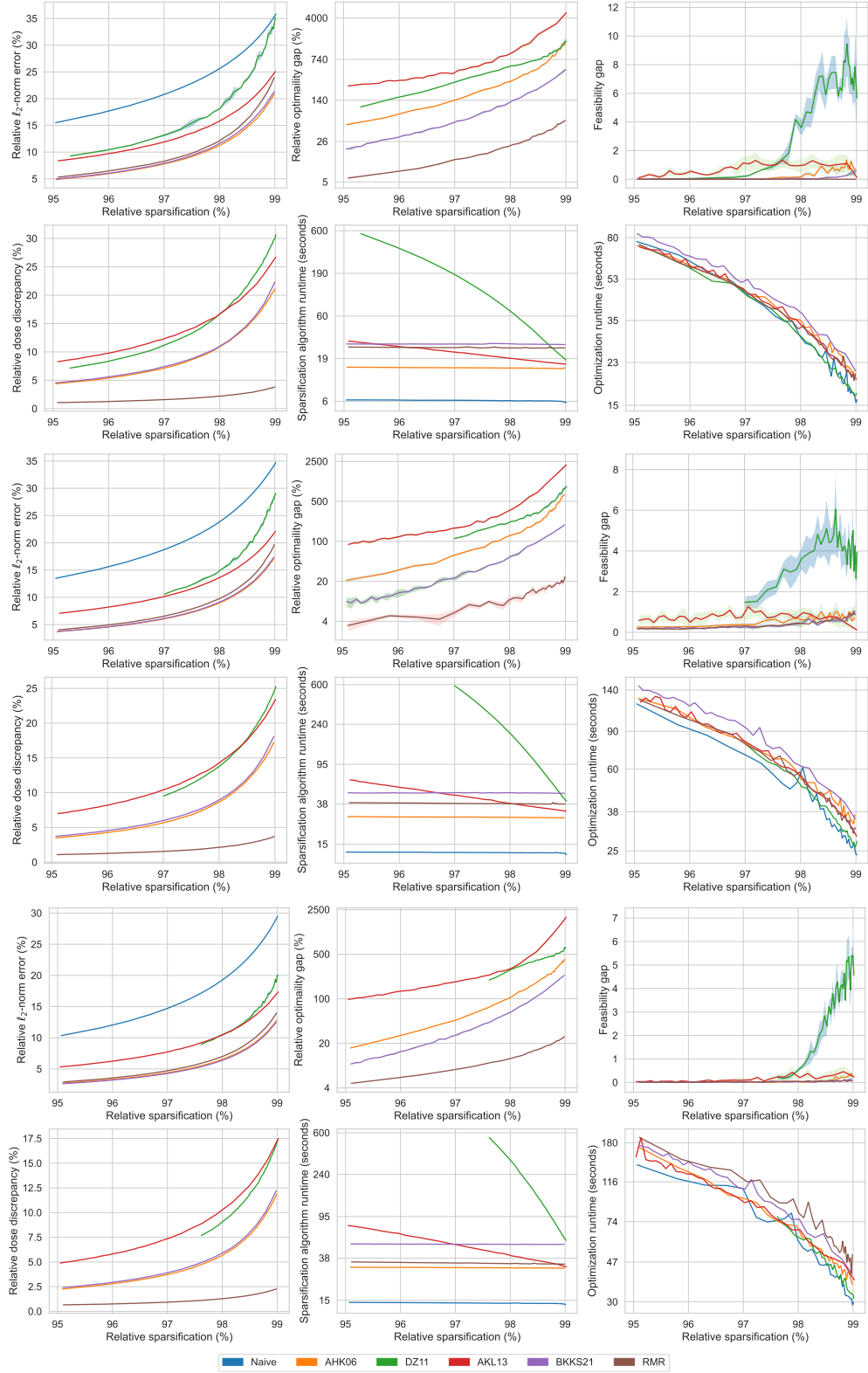

Figure 5: *(Various sparsification levels, Lung patients 5-7)* The performance of different algorithms in terms of: relative $\ell_2$-norm error, relative optimality gap, feasibility gap, relative dose discrepancy, algorithm runtime, and optimization runtime, for patients 5 (first two rows), 6 (second two rows), and 7 (third two rows).

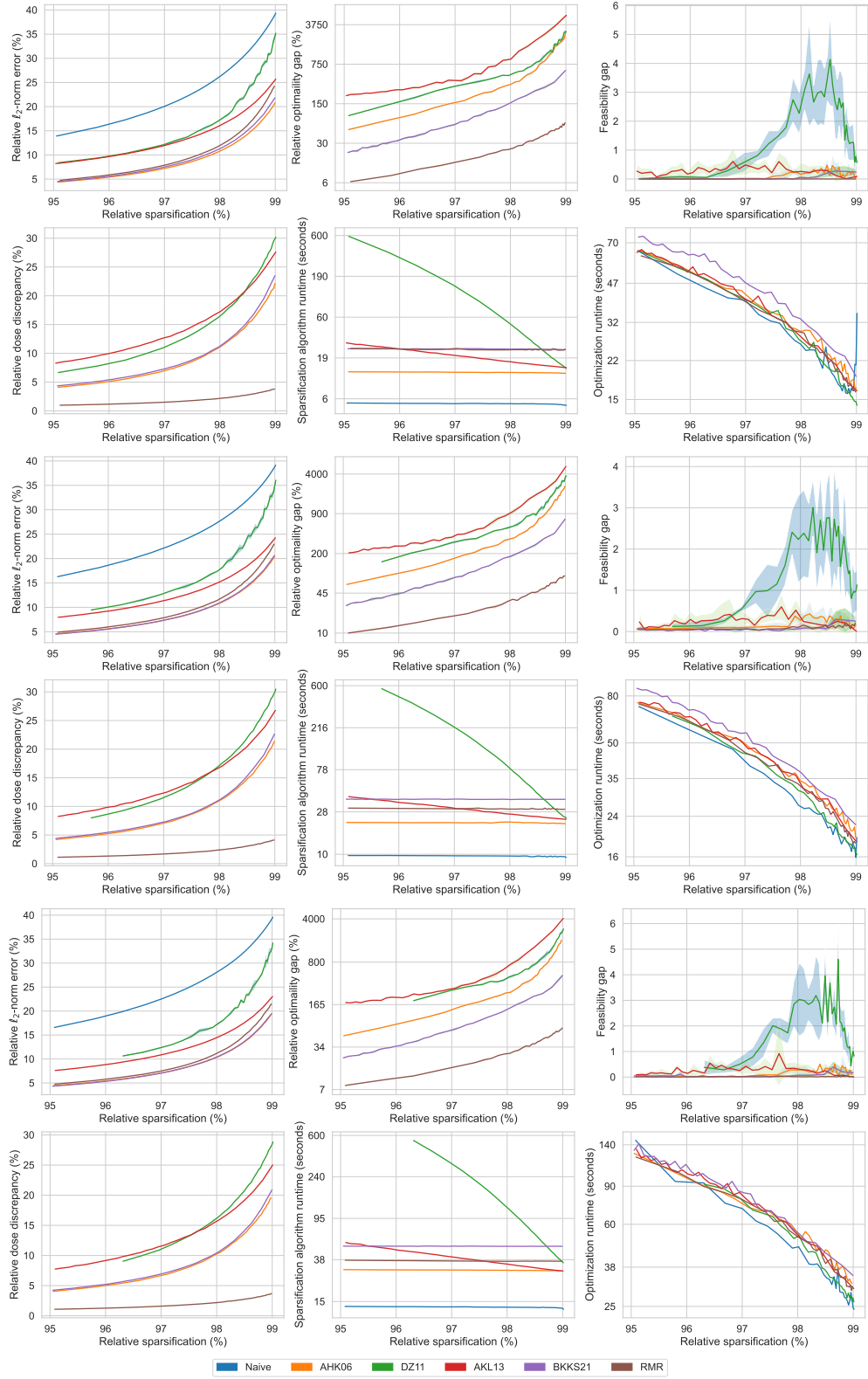

Figure 6: *(Various sparsification levels, Lung patients 8-10)* The performance of different algorithms in terms of: relative $\ell_2$-norm error, relative optimality gap, feasibility gap, relative dose discrepancy, algorithm runtime, and optimization runtime, for patients 8 (first two rows), 9 (second two rows), and 10 (third two rows).

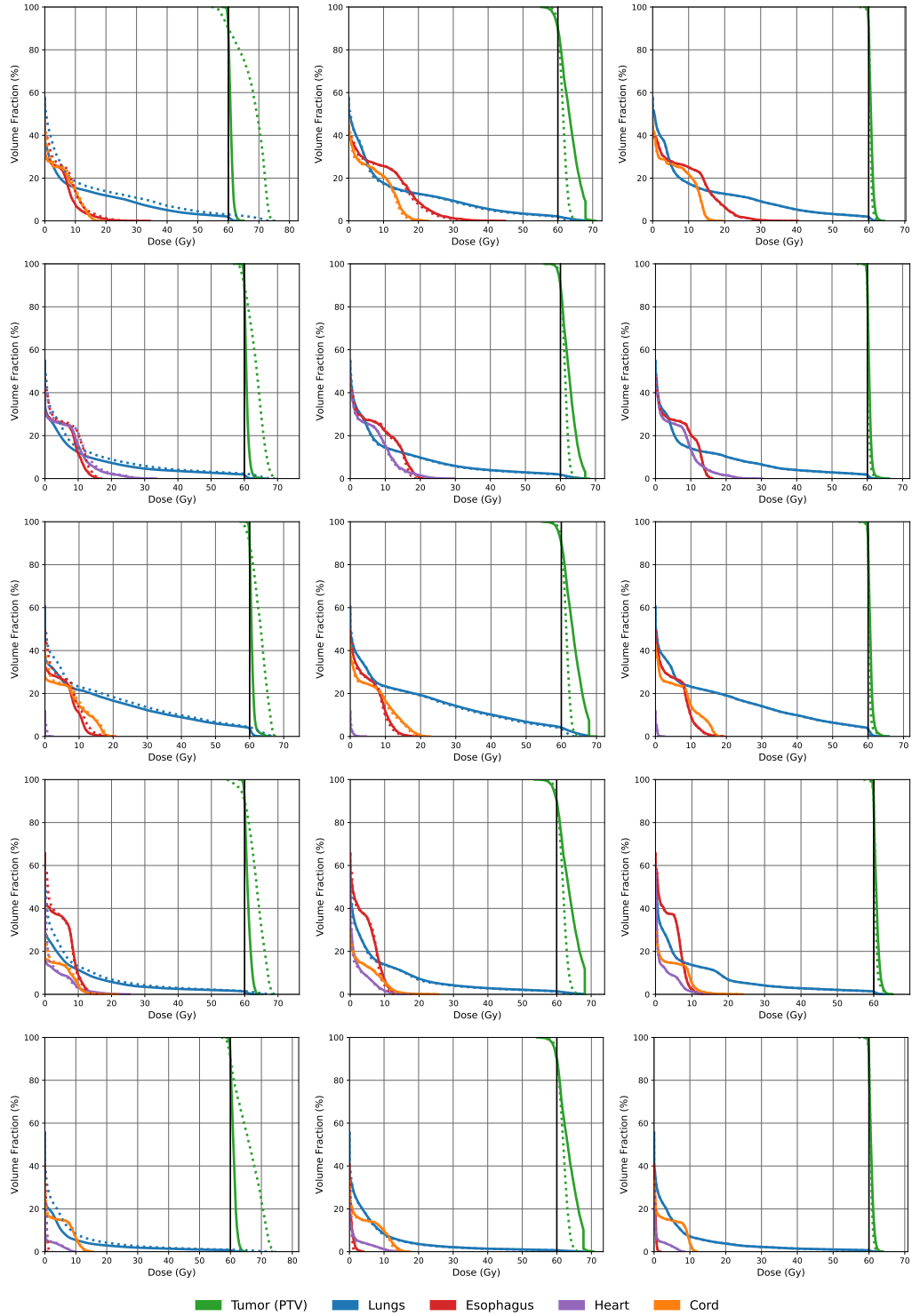

Figure 7: *(DVH discrepancies, Lung patients 1-5)* The discrepancies in Dose Volume Histogram (DVH) plots for patients 1 to 5, from top to bottom, respectively. The plots compare the results of the naive approach (left), AHK06 (middle), and the RMR method (right). In each plot, dashed lines represent the approximated radiation dose, $Sx_S$, while solid lines depict the actual radiation dose, $Ax_S$. A smaller gap between the dashed and solid lines is preferred, indicating a more accurate dose approximation. Note that for RMR, the solid and dashed lines often overlap, making the dashed lines nearly invisible.

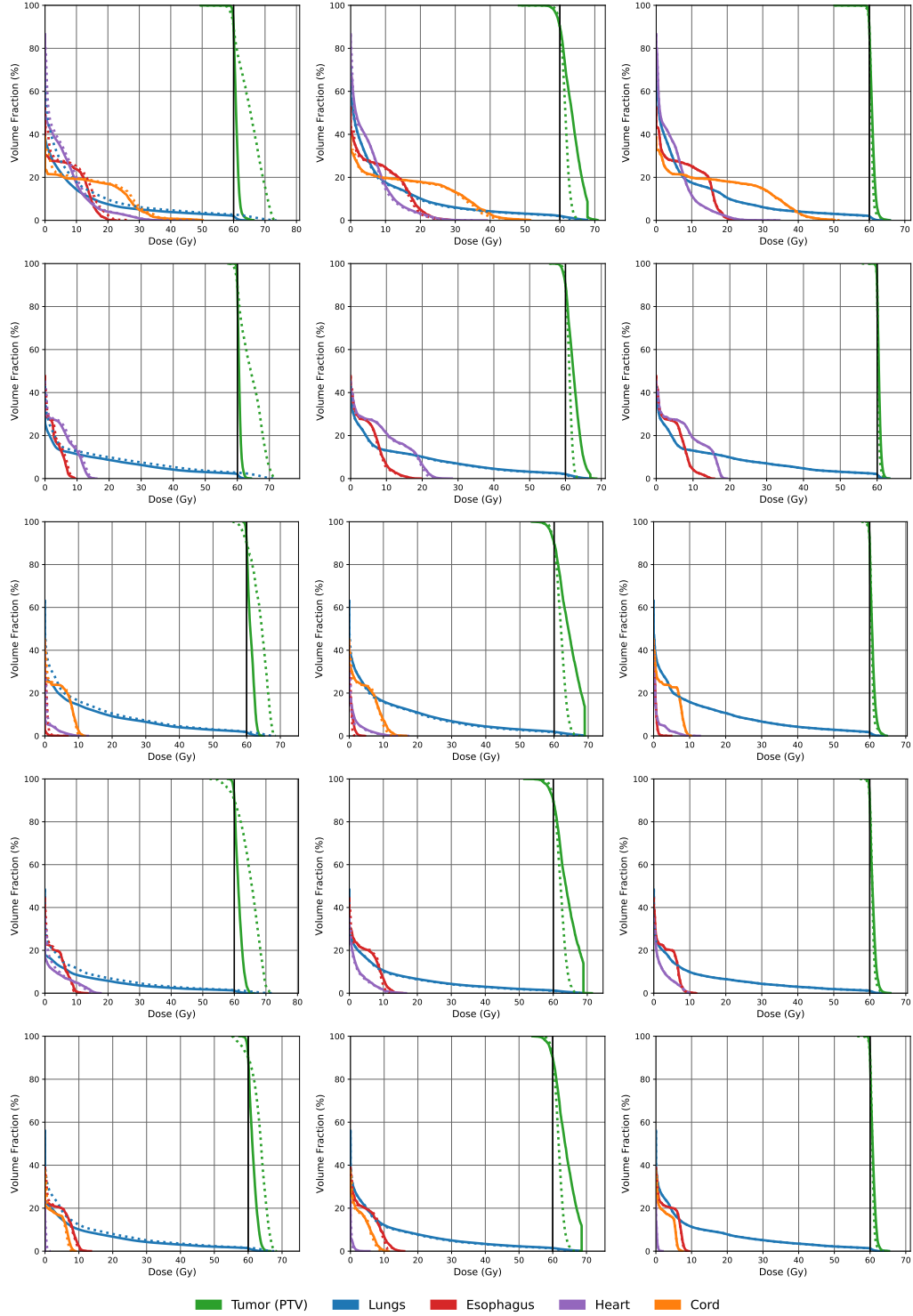

Figure 8: *(DVH discrepancies, Lung patients 6-10)* The discrepancies in Dose Volume Histogram (DVH) plots for patients 6 to 10, from top to bottom, respectively. The plots compare the results of the naive approach (left), AHK06 (middle), and the RMR method (right). In each plot, dashed lines represent the approximated radiation dose, $Sx_S$, while solid lines depict the actual radiation dose, $Ax_S$. A smaller gap between the dashed and solid lines is preferred, indicating a more accurate dose approximation. Note that for RMR, the solid and dashed lines often overlap, making the dashed lines nearly invisible.

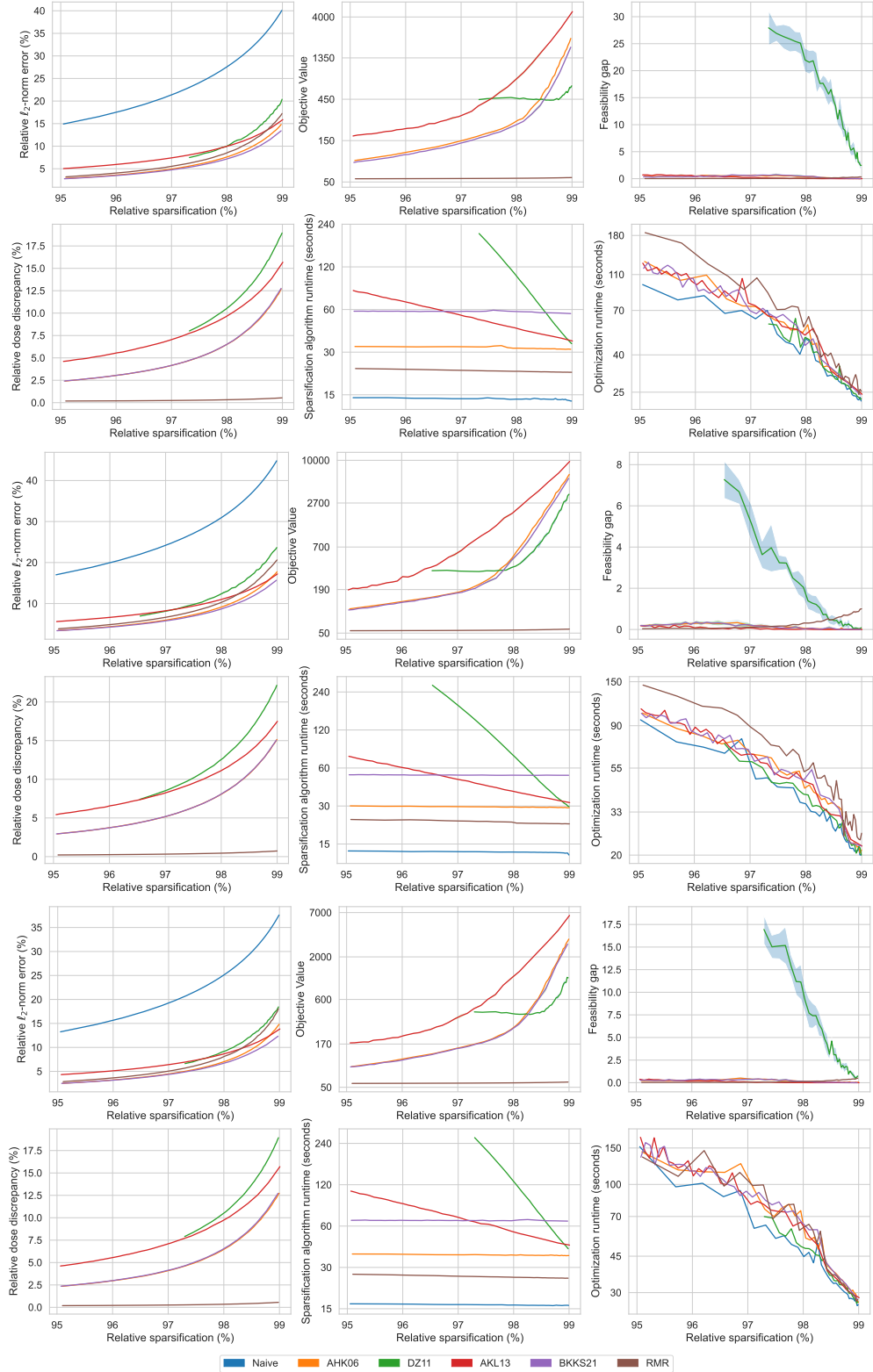

Figure 9: *(Various sparsification levels, Prostate patients 1-3)* The performance of different algorithms in terms of: relative $\ell_2$-norm error, relative optimality gap, feasibility gap, relative dose discrepancy, algorithm runtime, and optimization runtime, for patients 1 (first two rows), 2 (second two rows), and 3 (third two rows).

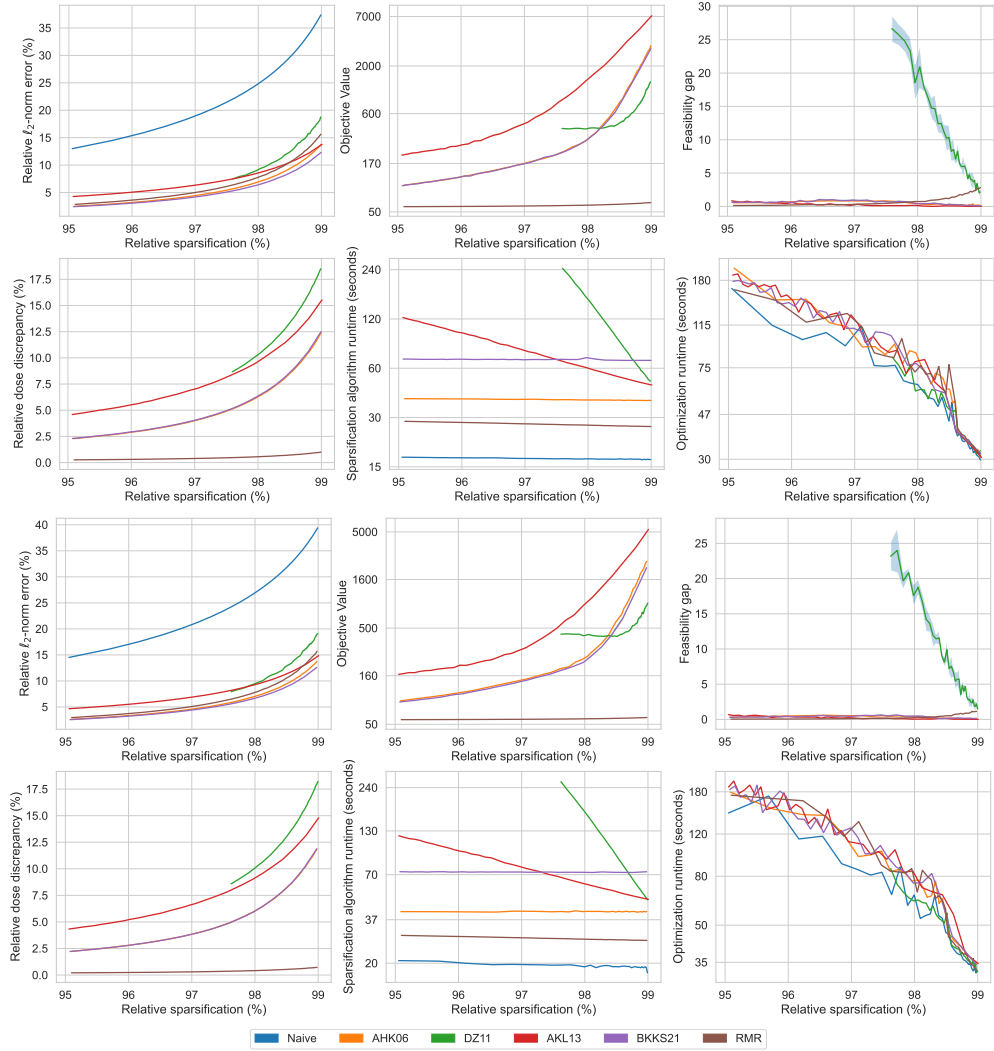

Figure 10: *(Various sparsification levels, Prostate patients 4-5)* The performance of different algorithms in terms of: relative $\ell_2$-norm error, relative optimality gap, feasibility gap, relative dose discrepancy, algorithm runtime, and optimization runtime, for patients 4 (first two rows), 5 (second two rows).

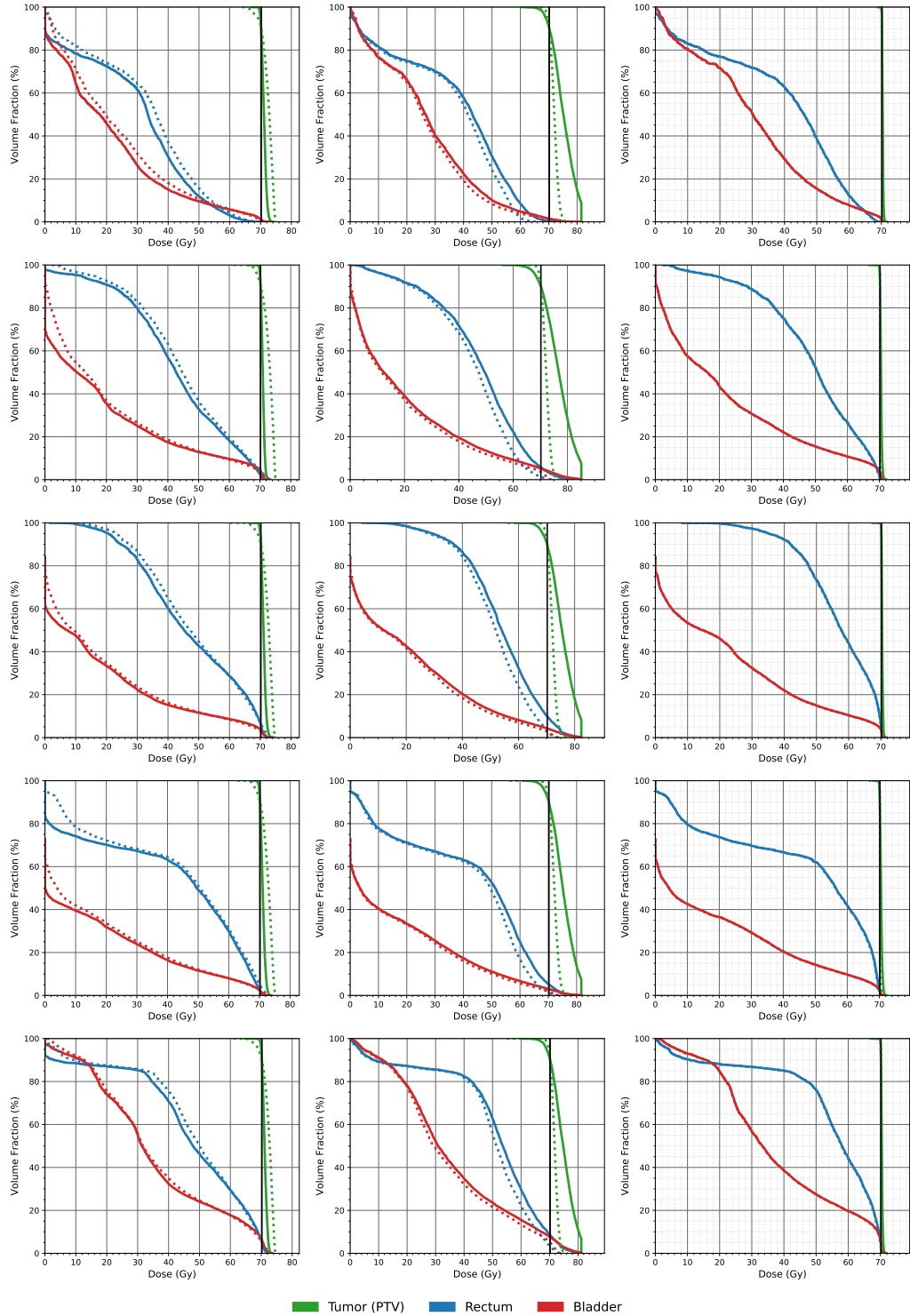

Figure 11: *(DVH discrepancies, Prostate patients 1-5)* The discrepancies in Dose Volume Histogram (DVH) plots for patients 1 to 5, from top to bottom, respectively. The plots compare the results of the naive approach (left), AHK06 (middle), and the RMR method (right). In each plot, dashed lines represent the approximated radiation dose, $Sx_S$, while solid lines depict the actual radiation dose, $Ax_S$. A smaller gap between the dashed and solid lines is preferred, indicating a more accurate dose approximation. Note that for RMR, the solid and dashed lines often overlap, making the dashed lines nearly invisible.

